# An Autoencoder-Like Nonnegative Matrix Co-Factorization for Improved Student Cognitive Modeling

**Shenbao Yu** [1]   **Yinghui Pan** [2*]   **Yifeng Zeng** [3]   **Prashant Doshi** [4]
**Guoquan Liu** [5]   **Kim-Leng Poh** [6]   **Mingwei Lin** [1]

[1] College of Computer and Cyber Security, Fujian Normal University, China
[2] National Engineering Laboratory for Big Data System Computing Technology,
Shenzhen University, China
[3] Department of Computer and Information Sciences, Northumbria University, UK
[4] Intelligent Thought and Action Lab, School of Computing, University of Georgia, USA
[5] Financial Technology Research Institute, Fudan University, China
[6] College of Design and Engineering, National University of Singapore, Singapore

{yushenbao,lmwfjnu}@fjnu.edu.cn, panyinghui@szu.edu.cn
yifeng.zeng@northumbria.ac.uk, pdoshi@uga.edu
liugq@fudan.edu.cn, pohkimleng@nus.edu.sg

## Abstract

Student cognitive modeling (SCM) is a fundamental task in intelligent education, with applications ranging from personalized learning to educational resource allocation. By exploiting students' response logs, SCM aims to predict their exercise performance as well as estimate knowledge proficiency in a subject. Data mining approaches such as matrix factorization can obtain high accuracy in predicting student performance on exercises, but the knowledge proficiency is unknown or poorly estimated. The situation is further exacerbated if only sparse interactions exist between exercises and students (or knowledge concepts). To solve this dilemma, we root monotonicity (a fundamental psychometric theory on educational assessments) in a co-factorization framework and present an autoencoder-like nonnegative matrix co-factorization (AE-NMCF), which improves the accuracy of estimating the student's knowledge proficiency via an encoder-decoder learning pipeline. The resulting estimation problem is nonconvex with nonnegative constraints. We introduce a projected gradient method based on block coordinate descent with *Lipschitz* constants and guarantee the method's theoretical convergence. Experiments on several real-world data sets demonstrate the efficacy of our approach in terms of both performance prediction accuracy and knowledge estimation ability, when compared with existing student cognitive models.

## 1   Introduction

With the explosion of open educational resources, student cognitive modeling is receiving growing attention. As illustrated in Figure 1, given a set of exercises (could be recommended by a learning platform) with the expert-annotated knowledge concepts in a subject domain, a student is required to finish the exercises and leaves the responses. Based on the response log, cognitive modeling aims to

---

(*a*) estimate the student's cognitive levels on the knowledge concepts (i.e., cognitive diagnosis) and (*b*) predict some exercise performance. With a comprehensive understanding of students, cognitive modeling is fruitful in applications such as computerized adaptive testing [1] and exercise recommendations [2]. To profile students' cognitive status, much progress has been made in the educational psychology area, where one popular avenue is to use cognitive diagnosis models (CDMs) [3]. While most CDMs provide detailed insights into students' cognitive states, the subjective handcraft features (e.g., the slip and guess of an exercise) may only partially capture the nuances of actual cognitive functioning, triggering cascading errors in predicting student performance [4].

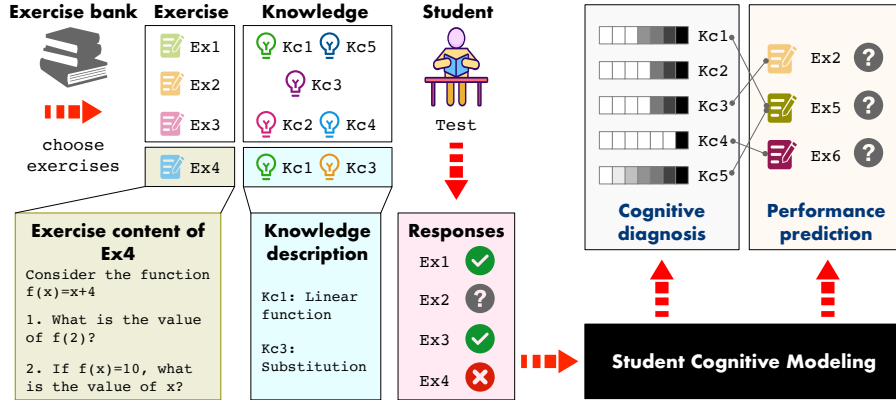

Figure 1: A schematic illustration of the student cognitive modeling problem. On the **left** is a set of exercises with the expert-labeled knowledge concepts. The **middle** is a student's binary-value response log with missing values (e.g., $Ex_2$ is missing) that is input to the modeling, and the **top right** illustrates the two cognitive tasks, which are the output of the modeling.

In a fresh direction, several studies focus on applying data mining techniques to model students' learning status, of which the cornerstone is matrix factorization (MF) [5]. By transforming students' response logs into a scoring matrix, MF-based models directly predict the missing response values via latent factors, thereby reducing cascading errors. In contrast to CDMs, MF-based models enjoy high prediction accuracy and are inexpensive to deploy [6]. On the other hand, the latent factors uncovered from the factorization techniques, which encode students' implicit learning ability, are unexplainable, i.e., the true knowledge components of the students remain unknown in the latent vectors. Recognizing this problem, a follow-up scalable nonnegative matrix co-factorization (SNMCF) model [4] utilizes a point coverage function to learn students' proficiency levels via pre-trained latent factors. However, SNMCF solves the two learning tasks separately, i.e., the generation of latent features is aimed at performance prediction without considering the target of improving cognitive diagnosis, thereby compromising the identification of student cognitive levels. As such, the fundamental issue of identifying students' knowledge proficiency remains an open problem.

In this paper, we envision a reliable and interpretable data mining-based cognitive model with interlocking learning components. Learning latent factors that help pinpoint students' responses to exercises can guide the assessment of their knowledge proficiency, and the corresponding latent knowledge features, in turn, enable their success or failure on the exercises. To this end, several challenges exist: How can we specify and assess the students' knowledge proficiency since the ground truth of the cognitive levels is unknown [1]? How can we frame the two learning tasks as the building blocks of an optimization framework while reducing cascading errors?

To mitigate these challenges, we leverage the known monotonicity [7] to sidestep the issue of unknown knowledge proficiency. The monotonicity states that a student's knowledge proficiency has a monotonic relationship with the probability of the right responses to related exercises. Furthermore, by investigating the form of an autoencoder, our key observation reveals that its self-reconstruction principle, which aims to reconstruct input data from the learned low-dimensional representations, is amenable to the requirement of the monotonic constraint. Leveraging this observation, we root the monotonicity in a co-factorization framework via the autoencoder mechanism. Consequently, an autoencoder-like nonnegative matrix co-factorization (AE-NMCF) is presented, which enables an iterative link between students' knowledge proficiency and exercise performance, thereby enhancing prediction accuracy and diagnostic ability. As the resulting optimization problem is not convex and

has nonnegative constraints – which makes the complexity acute by an inverse link function (often called response functions in the case of general linear models [8]) – we develop a projected gradient method based on block coordinate descent with *Lipschitz* constants and guarantee its theoretical convergence. The main contributions are:

- We introduce a data mining-based model (AE-NMCF) for improved student cognitive modeling, which provides an end-to-end and data-driven way of specifying and assessing students' understanding of a set of given knowledge concepts. This new model exploits the monotonicity in educational MF-based approaches for the first time.

- To learn the model, we present a novel projected gradient method based on block coordinate descent with *Lipschitz* constants, for which theoretical convergence is guaranteed. This method accounts for the non-convexity of the optimization function with nonnegative constraints and the complexity of the inverse link function.

- AE-NMCF provides a good fit to the students' knowledge proficiency while maintaining student performance prediction that is comparable to other student cognitive models.

These contributions will potentially improve the automated comprehensive understanding of students' knowledge learning and benefit numerous intelligent educational tools.

## 2 Related Works

**Student cognitive modeling has generally proceeded along two tracks: cognitive diagnostic models (CDMs) and data mining approaches.** CDMs are of two types: continuous CDMs, an example of which is item response theory (IRT) [9]), and discrete CDMs such as the deterministic inputs, noisy "and" gate (DINA) [10]). IRT predicts a student's exercise performance based on a single learning trait and exercise difficulty levels. In contrast, DINA probes a student's binary cognitive status in different knowledge concepts, which assumes that a student can answer correctly if she has mastered all the required knowledge concepts. Traditional CDMs engender a plethora of advanced models. For example, Cheng *et al.* [11] extend IRT using deep learning to enhance the diagnostic process. Noting the importance of the relation among knowledge concepts, Gao *et al.* [12] proposed a deep diagnosis framework that considers both the importance of and the interactions between knowledge concepts. Furthermore, Yang *et al.* [13] recently presented a relationship-based CDM to explore implicit knowledge-exercise relations that educators ignore.

Along the data mining approach, MF has proven to be effective in understanding students' response processes [12], especially toward student performance prediction. In this study, classic models (e.g., nonnegative MF (NMF) [14]) and their variants such as the regularized NMF [15] were successfully applied. Because the latent trait of MF is not interpretable for knowledge estimation, Yu *et al.* [4] proposed SNMCF that utilizes a coverage function to model students' knowledge states, thereby taking an important stride in data mining-based student cognitive modeling. But, the coverage function often gives binary cognitive levels, failing to discern the nuance between knowledge proficiencies.

**Matrix co-factorization (MCF) [16] benefits from jointly exploiting multiple data sources.** It is well established for many applications such as convolutive source separation [17], data sparsity [18, 19], and decision support systems [20]. Given a domain task, MCF improves performance by incorporating knowledge in additional matrices (e.g., trust relationship for social recommendation [21]), which share latent factors with the original one. This sharing mechanism facilitates entity-relation learning [22]. It motivates us to develop an in-depth understanding of students, exercises, and knowledge concepts, and facilitates an effective solution for the two learning tasks, which differs from existing MCF-based approaches that aim at performance boost only.

**Recently, the autoencoder architecture is being explored in dimensionality reduction [23], classification [24], and anomaly detection [25, 26].** For example, Wang *et al.* [23] proposed a deep version of autoencoder to explore manifold data structures. Gong *et al.* [25] augmented the autoencoder with a memory module to mitigate anomaly reconstruction problems. For student cognitive modeling, given the reconstruction ability of autoencoder, our work is the first attempt to exploit this mechanism in MF-based approaches to estimate student knowledge proficiency.

# 3 AE-NMCF Model and Method

Given M students and N exercises, all students' responses to the exercises are recorded in a binary scoring matrix $\mathbf{X} \in [0|1]^{\mathrm{N} \times \mathrm{M}}$, where $\mathbf{X}_{nm}$ denotes student $\mathrm{St}_m$'s answer on exercise $\mathrm{Ex}_n$. In addition, given K knowledge concepts, we have an expert-labeled Q-matrix $\mathbf{Q} \in [0|1]^{\mathrm{N} \times \mathrm{K}}$, where $\mathbf{Q}_{nk} = 1$ if $\mathrm{Ex}_n$ relates to knowledge concept $\mathrm{Kc}_k$, otherwise $\mathbf{Q}_{nk} = 0$. With $\mathbf{X}$ and $\mathbf{Q}$ in hand, we aim to $(a)$ learn students' proficiency in knowledge concepts from the responses, and $(b)$ predict students' performance on exercises that they have never done.

## 3.1 Model Formulation

Figure 2 (from left to right) offers an overview of the approach, which includes an encoder and a decoder. The encoder and decoder specify and diagnose students' cognitive levels, thereby enabling monotonicity. Specifically, the new framework receives the student scoring matrix ($\mathbf{X}$) and the Q-matrix ($\mathbf{Q}$). In the encoder process, we introduce the exercise-knowledge association matrix ($\mathbf{B}$) and then jointly decompose $\mathbf{X}$ and $\mathbf{B}$ to obtain three low-dimensional nonnegative matrices: the student proficiency matrix ($\mathbf{U}$), the exercise characteristic matrix ($\mathbf{E}$), and the knowledge requirement matrix ($\mathbf{V}$). Note that the shared matrix $\mathbf{E}$ places $\mathbf{U}$ on the same scale as $\mathbf{V}$, which shapes a pathway to specify the students' knowledge proficiency ($\mathbf{A}$). In the decoder process, we introduce the exercise difficulty vector ($\mathbf{M}$), which is combined with $\mathbf{A}$ and $\mathbf{B}$ to form cognitive factors. By re-fitting $\mathbf{X}$, the decoder process ensures that students' knowledge proficiency is monotonic with the probability of the correct exercise responses, which embodies our desire to maintain the monotonicity.

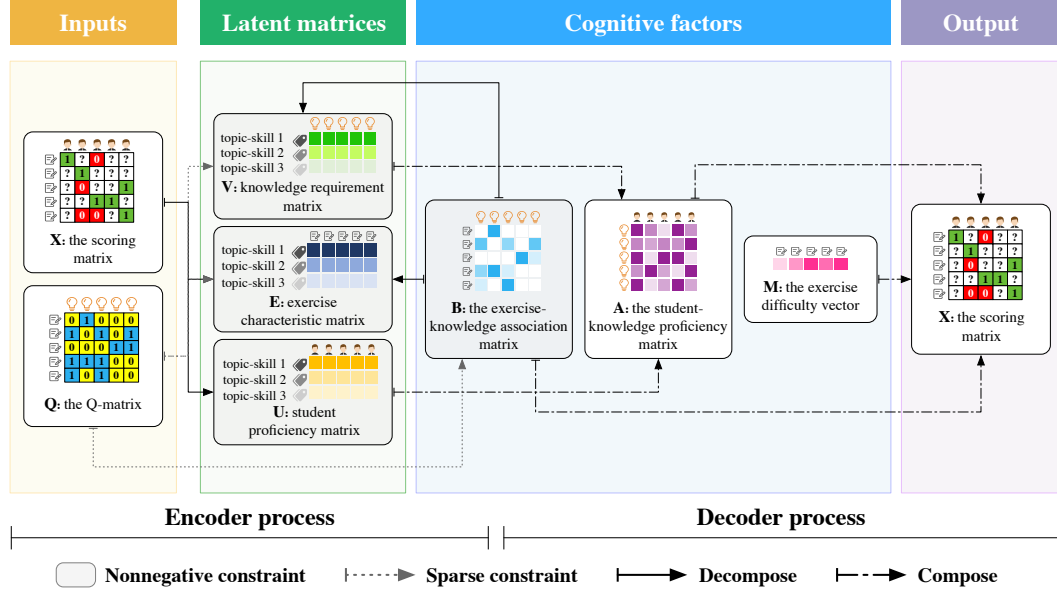

Figure 2: The end-to-end pipeline of AE-NMCF. We start from the scoring matrix ($\mathbf{X}$), which is also the ending module. The question marks ('?') in $\mathbf{X}$ denote the absent responses that the students have never visited the exercises before. Here, we use the cell shadings to highlight the nonnegative constraints on the matrix blocks, wherein the dotted lines impose the sparse constraints. In addition, the solid and chain-dotted lines denote the decomposing and composing processes, respectively.

**The encoder process.** Given $\mathbf{X} \in [0|1]^{\mathrm{N} \times \mathrm{M}}$ and $\mathbf{Q} \in [0|1]^{\mathrm{N} \times \mathrm{K}}$, we start the encoder process with optimization problem (1), where we have three low-dimensional nonnegative matrices: $\mathbf{E} \in \mathbb{R}^{\mathrm{N} \times \mathrm{T}}$, $\mathbf{U} \in \mathbb{R}^{\mathrm{T} \times \mathrm{M}}$, and $\mathbf{V} \in \mathbb{R}^{\mathrm{T} \times \mathrm{K}}$, each of which consists of T latent factors. The latent factors can be loosely viewed as a series of topic skills denoting high-level knowledge in a subject area, such as "spatial imagination" and "abstract summarization" in mathematics.

$$\min_{\mathbf{B},\mathbf{U},\mathbf{E},\mathbf{V}} \|\mathbf{W} \odot (\mathbf{X} - \mathbf{E}\mathbf{U})\|_{\mathrm{F}}^2 + \|\mathbf{Q} \odot (\mathbf{B} - \mathbf{E}\mathbf{V})\|_{\mathrm{F}}^2$$
$$\text{s.t.} \quad \mathbf{B} \geq \mathbf{0}, \mathbf{U} \geq \mathbf{0}, \mathbf{E} \geq \mathbf{0}, \mathbf{V} \geq \mathbf{0}, \tag{1}$$

For the first term in problem (1), we use $\mathbf{E}$ and $\mathbf{U}$ to approximate $\mathbf{X}$ through the *Frobenius* norm and introduce a weighted matrix $\mathbf{W} \in [0|1]^{\mathrm{N} \times \mathrm{M}}$ to focus on the observed entries in $\mathbf{X}$ via the Hadama product ($\odot$). In the second term, considering that $\mathbf{Q}$ only stores the linkage between exercises and knowledge concepts with either true or false relations (failing to uncover their strength), we introduce the nonnegative matrix $\mathbf{B} \in \mathbb{R}^{\mathrm{N} \times \mathrm{K}}$, where $\mathbf{B}_{nk}$ is the degree to which exercise $\mathrm{Ex}_n$ involves knowledge concept $\mathrm{Kc}_k$, with larger values denoting stronger involvement of the knowledge concept. Similarly, we use $\mathbf{E}$ and $\mathbf{V}$ to approximate $\mathbf{B}$, where the sparsity is imposed by $\mathbf{Q}$.

In problem (1), $\mathbf{X}$ and $\mathbf{B}$ share the matrix $\mathbf{E}$, which bridges the gap between students and knowledge concepts. In reality, given $\mathbf{E}$, the approximation for $\mathbf{X}_{:m}$ is a linear accumulation of the columns of $\mathbf{E}$, weighted by the components of $\mathbf{U}_{:m}$, and so does $\mathbf{B}_{:k}$: we project the two nonnegative vectors $\mathbf{U}_{:m}$ and $\mathbf{V}_{:k}$ into the new basis $\mathbf{E}$ [4]. Since the latent factors are considered topic skills [27], we define $\mathbf{U}_{tm}$ as the *topic knowledge* of student $\mathrm{St}_m$ on $t$-th topic skill, as well as $\mathbf{V}_{tk}$ as the *topic requirement* of $\mathrm{Kc}_k$ accordingly. Based on $\mathbf{U}_{tm}$ and $\mathbf{V}_{tk}$, we specify students' knowledge proficiency via the matrix $\mathbf{A} = \mathbf{V}^\top \mathbf{U} \in \mathbb{R}^{\mathrm{K} \times \mathrm{M}}$, where $\mathbf{A}_{km}$ is the cognitive level of $\mathrm{St}_m$ on $\mathrm{Kc}_k$.

**The decoder process.** Recall that the matrix $\mathbf{A}$ specified by problem (1) does not give an off-the-shelf diagnostic solution due to the ignorance of monotonic constraints. We remedy this void by reconstructing the scoring matrix $\mathbf{X}$. Specifically, we first assume that exercise $\mathrm{Ex}_n$ has an intrinsic difficulty level $\mu_n \in \mathbb{R}$, which are stacked into a column vector $\mathbf{M} = [\mu_1, \mu_2, \cdots, \mu_\mathrm{N}]^\top$. Armed with $\mathbf{A}$, $\mathbf{B}$, and $\mathbf{M}$, the probability that $\mathrm{St}_m$ answers $\mathrm{Ex}_n$ correctly is

$$\Phi(\Delta_{nm}) = \int_{-\infty}^{\Delta_{nm}} \mathcal{N}(t)\mathrm{d}t = \frac{1}{\sqrt{2\pi}} \int_{-\infty}^{\Delta_{nm}} \mathrm{e}^{-t^2/2}\mathrm{d}t, \tag{2}$$

where $\Delta_{nm} = \mathbf{B}_{n:}\mathbf{A}_{:m} + \mu_n$ indicates that $\mathrm{St}_m$'s response to $\mathrm{Ex}_n$ is generated by a linear accumulation of required knowledge concepts. In addition, we use an *inverse link function* $\Phi(x)$, which is often a response function in generalized linear models, to map $\Delta_{nm}$ to the success probability of the binary response $\mathbf{X}_{nm}$. $\Phi(x)$ can be any monotonic differentiable function. Here, we focus on the commonly used *probit link function* with the probability density of the standard Gaussian distribution.

Given Eq. (2), we can maximize the likelihood of the observed data $\mathbf{X}_{nm}$ as

$$\mathrm{Pr}(\mathbf{X}_{nm}) = \Phi(\Delta_{nm})^{\mathbf{X}_{nm}} \left[1 - \Phi(\Delta_{nm})\right]^{(1-\mathbf{X}_{nm})}, \tag{3}$$

and the likelihood finally yields the following optimization problem

$$\min_{\mathbf{B}_{n:}, \mathbf{A}_{:m}, \mu_n : \forall n, m} -\ell + \frac{\gamma}{2} \sum_{n=1}^{\mathrm{N}} \|\mathbf{B}_{n:}\|_2^2, \tag{4}$$

where $\ell = \sum_{(n,m) \in \Omega_\mathrm{o}} \log \mathrm{Pr}(\mathbf{X}_{nm})$ is the log-likelihood term, and $\Omega_\mathrm{o} \subseteq \{1, \cdots, \mathrm{N}\} \times \{1, \cdots, \mathrm{M}\}$ contains indices of the observed responses in $\mathbf{X}$. In addition, since one can arbitrarily increase the scale of the vector $\mathbf{B}_{n:}$ while decreasing the scale of the vector $\mathbf{A}_{:m}$ (or $\mathbf{V}^\top \mathbf{U}_{:m}$) accordingly (and vice versa) without changing the likelihood, we gauge the vector $\mathbf{B}_{n:}$ using the regularization term $\sum_{n=1}^{\mathrm{N}} \|\mathbf{B}_{n:}\|_2^2$ with the regularization parameter $\gamma > 0$. To illustrate the encoder-decoder process further, we provide an example in Appendix A.

**Objective function.** By combining the encoder and decoder, the objective function ($\mathcal{O}_{\mathrm{AF}}$) is

$$\min_{\mathbf{B}, \mathbf{U}, \mathbf{E}, \mathbf{V}, \mathbf{M}} \quad \mathcal{O}_{\mathrm{AF}} = -\ell + \|\mathbf{W} \odot (\mathbf{X} - \mathbf{EU})\|_{\mathrm{F}}^2 + \|\mathbf{Q} \odot (\mathbf{B} - \mathbf{EV})\|_{\mathrm{F}}^2 + \frac{\gamma}{2} \sum_{n=1}^{\mathrm{N}} \|\mathbf{B}_{n:}\|_2^2, \tag{5}$$

$$\text{s.t.} \quad \mathbf{B} \geq \mathbf{0}, \mathbf{U} \geq \mathbf{0}, \mathbf{E} \geq \mathbf{0}, \mathbf{V} \geq \mathbf{0}.$$

It is worth taking a few moments to study the form of problem (5) as it enables the monotonicity from two viewpoints. First, the monotonicity is achieved by the monotonic formulation in Eq. (2); Second, the monotonicity is optimized by problem (4). They jointly guarantee that a large value of knowledge proficiency corresponds to a better chance of success on related exercises.

## 3.2  Model Solution

In problem (5), the first term $-\ell$ is convex for the probit link function [28]. The second and third terms are convex in either $\mathbf{B}$, $\mathbf{U}$, $\mathbf{E}$, or $\mathbf{V}$ only, but they are not convex in all the variables together.

Given the nonnegative constraints, we employ the projected gradient (PG) method [29]. Furthermore, concerning blocks of $\mathbf{B}_{n:}$ and $\mathbf{U}_{:m}$, we apply the PG via a block coordinate descent (PG-BCD) approach. Hence, problem (5) can be expressed in a block fashion as

$$\min_{\mathbf{B},\mathbf{U},\mathbf{E},\mathbf{V},\mathbf{M}} \mathcal{O}_{\mathrm{AF}} = -\ell + \sum_{m=1}^{\mathrm{M}} \|\mathbf{W}_{:m} \odot (\mathbf{X}_{:m} - \mathbf{E}\mathbf{U}_{:m})\|_2^2 + \sum_{n=1}^{\mathrm{N}} \|\mathbf{Q}_{n:} \odot (\mathbf{B}_{n:} - \mathbf{E}_{n:}\mathbf{V})\|_2^2$$

$$+ \frac{\gamma}{2} \sum_{n=1}^{\mathrm{N}} \|\mathbf{B}_{n:}\|_2^2, \qquad \text{s.t. } \mathbf{B} \geq \mathbf{0}, \mathbf{U} \geq \mathbf{0}, \mathbf{E} \geq \mathbf{0}, \mathbf{V} \geq \mathbf{0}.$$

Accordingly, the subproblems of $\mathbf{B}_{n:}$, $\mathbf{U}_{:m}$, $\mathbf{E}$, $\mathbf{V}$, and $\mu_n$ constitute the iterations of PG-BCD for AE-NMCF. Next, we show the parameter solution for $\mathbf{B}_{n:}$ in problem (6) below. For further details on the remaining parameters, refer to Appendix B.

$$\min_{\mathbf{B}_{n:} \geq \mathbf{0}} \mathcal{O}_{\mathrm{AF}}(\mathbf{B}_{n:}) = \sum_m - \log \Pr(\mathbf{X}_{nm}) + \frac{\gamma}{2}\|\mathbf{B}_{n:}\|_2^2 + \|\mathbf{Q}_{n:} \odot (\mathbf{B}_{n:} - \mathbf{E}_{n:}\mathbf{V})\|_2^2. \tag{6}$$

To solve problem (6), we note that second-order methods do not scale well to high-dimensional problems due to the necessary computation of the Hessian, making explicit calculation difficult for the probit link function. Thus, we build our learning algorithm on first-order methods. To do so, we first derive the gradients of $\mathcal{O}_{\mathrm{AF}}(\mathbf{B}_{n:})$ as

$$\nabla\mathcal{O}_{\mathrm{AF}}(\mathbf{B}_{n:}) = -\sum_m \Xi_{nm}[\mathbf{X}_{nm} - \Phi(\Delta_{nm})]\mathbf{U}_{:m}^\top\mathbf{V} + 2[\mathbf{Q}_{n:}\odot\mathbf{B}_{n:} - \mathbf{Q}_{n:}\odot(\mathbf{E}_{n:}\mathbf{V})] + \gamma\mathbf{B}_{n:},$$

where $\Xi_{nm} = \frac{\mathcal{N}(\Delta_{nm})}{\Phi(\Delta_{nm})[1-\Phi(\Delta_{nm})]}$, and we can employ the gradients above to search for the optimum point. In each iteration $l = 1, 2, \cdots$, the gradient step is

$$\mathbf{B}_{n:}^{(l+1)} \leftarrow \left[\mathbf{B}_{n:}^{(l)} - \eta_{\mathbf{B}_{n:}}^{(l)} \nabla\mathcal{O}_{\mathrm{AF}}^{(l)}(\mathbf{B}_{n:})\right]_+, \tag{7}$$

where the half-wave rectifier $[x]_+ = max(\kappa, x)$, $\kappa = 10^{-15}$ ensures the nonnegativity [30], and $\eta_{\mathbf{B}_{n:}}^{(l)}$ is a suitable step size. For Eq. (7), a key issue is to choose the appropriate step size $\eta_{\mathbf{B}_{n:}}^{(l)}$, and a simple strategy is "*Armijo* rule along the projection arc" [31]. Although the convergence is guaranteed, it is time-consuming to search for feasible values. Motivated by Lan *et al.* [6], we determine the appropriate step sizes by *Lipschitz* constants [32]. A common approach that guarantees convergence of a function $f$ is to set $\eta^{(l)} = 1/L$, where $L$ is the *Lipschitz* constant of $\nabla f$.

### 3.3 Algorithm and Theoretical Analysis

We start with Lemma 1 [6] to analyze the *Lipschitz* constant for problem (6). After that, we present the parameter learning algorithm for problem (5) and conclude with its theoretical analysis.

**Lemma 1.** *Let $g(x) = \frac{\Phi'(x)}{\Phi(x)}$, $x \in \mathbb{R}$, where $\Phi(x)$ is the probit link function. Then, for $y, z \in \mathbb{R}$, we have $|g(y) - g(x)| \leq L_p|y - z|$. Here, $L_p = 1$ is the scalar Lipschitz constant of $g(x)$.*

Since Eq. (3) can be rewritten as $\Pr(\mathbf{X}_{nm}) = \Phi\left((2\mathbf{X}_{nm} - 1)(\mathbf{B}_{n:}\mathbf{V}^\top\mathbf{U}_{:m} + \mu_n)\right)$ for $\Phi(\cdot)$, we derive the following theorem which serves as a bound on the (vector) *Lipschitz* constant for problem (6), using the result in Lemma 1.

**Theorem 1.** *For a given $n$, substituting $\Pr(\mathbf{X}_{nm})$ in problem (6) with the right hand side expression above yields the following*

$$\mathcal{O}_{AF}(\mathbf{B}_{n:}) = \sum_m - \log \Phi(\Lambda_{nm}) + \frac{\gamma}{2}\|\mathbf{B}_{n:}\|_2^2 + \|\mathbf{Q}_{n:} \odot (\mathbf{B}_{n:} - \mathbf{E}_{n:}\mathbf{V})\|_2^2,$$

*where $\Lambda_{nm} = (2\mathbf{X}_{nm} - 1)(\mathbf{B}_{n:}\mathbf{V}^\top\mathbf{U}_{:m} + \mu_n)$. For any vectors $\mathbf{y}, \mathbf{z}$, we have*

$$\|\nabla\mathcal{O}_{AF}(\mathbf{y}) - \nabla\mathcal{O}_{AF}(\mathbf{z})\|_2 \leq \left[L_p\sigma_1^2(\mathbf{U}^\top\mathbf{V}) + 2\left(\sum_{k=1}^{\mathrm{K}} \mathbf{Q}_{nk}^2\right)^{\frac{1}{2}} + \gamma\right]\|\mathbf{y} - \mathbf{z}\|_2.$$

To prove Theorem 1, we first derive the gradient of $\mathcal{O}_{\text{AF}}(\mathbf{B}_{n:})$ based on the element-wise operation of $\mathcal{N}(\cdot)$ and $\Phi(\cdot)$. After that, we establish the upper bound of the $\ell_2$-norm of the gradient difference given two arbitrary points $\mathbf{y}$ and $\mathbf{z}$. The derivation details refer to Appendix D. By comparing with Theorem 1, we obtain the *Lipschitz* constant as $L_p \sigma_1^2(\mathbf{U}^\top \mathbf{V}) + 2 \left( \sum_{k=1}^{K} \mathbf{Q}_{nk}^2 \right)^{\frac{1}{2}} + \gamma$, where $\sigma_1(\cdot)$ denotes the corresponding maximum singular value.

Armed with Eq. (7) with the step sizes determined by Theorem 1, Algorithm 1 outlines the optimization process for AE-NMCF, named PG-BCD+*Lipschitz*. In Algorithm 1, we first initialize all parameters with random entries (line 1) and then optimize $\mathcal{O}_{\text{AF}}$ in an alternating fashion. Each outer iteration solves the inner subproblems (lines 3 − 9). For each subproblem, we optimize the target parameter and hold others constant. For example, we hold $\mathbf{U}$, $\mathbf{E}$, $\mathbf{V}$, and $\mathbf{M}$ constant and separately optimize each block of variables in $\mathbf{B}$. The update order in the block case is $\mathbf{B}_{1:} \to \mathbf{B}_{2:} \to \cdots \to \mathbf{B}_{N:}$. The outer loop is terminated if the decrease in $\mathcal{O}_{\text{AF}}$ is smaller than a threshold $\epsilon$ (lines 6 − 8).

---

**Algorithm 1** PG-BCD+*Lipschitz*

---

**Input:** $\mathbf{X}$, $\mathbf{Q}$, and $\epsilon$.
**Output:** $\mathbf{B}_{n:}$, $\mathbf{U}_{:m}$, $\mathbf{E}$, $\mathbf{V}$, and $\mu_n$ ($1 \leq n \leq N, 1 \leq m \leq M$).
1: Initialize $\mathbf{B}_{n:}^{(0)} \geq \mathbf{0}$, $\mathbf{U}_{:m}^{(0)} \geq \mathbf{0}$, $\mathbf{E}^{(0)} \geq \mathbf{0}$, $\mathbf{V}^{(0)} \geq \mathbf{0}$, and $\mu_n^{(0)}$ ($1 \leq n \leq N, 1 \leq m \leq M$).
2: Calculate $\mathcal{O}_{\text{AF}}^{(0)}$.
3: **for** $l = 0, 1, \cdots$ **do**
4:     Update: $\mathbf{B}_{n:}^{(l+1)}$ ($1 \leq n \leq N$); $\mathbf{U}_{:m}^{(l+1)}$ ($1 \leq m \leq M$); $\mathbf{E}^{(l+1)}$; $\mathbf{V}^{(l+1)}$; $\mu_n^{(l+1)}$ ($1 \leq n \leq N$).
5:     Calculate $\mathcal{O}_{\text{AF}}^{(l+1)}$.
6:     **if** $|\mathcal{O}_{\text{AF}}^{(l+1)} - \mathcal{O}_{\text{AF}}^{(l)}| \leq \epsilon$ **then**
7:         Return $\mathbf{B}_{n:}^{(l+1)}$, $\mathbf{U}_{:m}^{(l+1)}$, $\mathbf{E}^{(l+1)}$, $\mathbf{V}^{(l+1)}$, and $\mu_n^{(l+1)}$.
8:     **end if**
9: **end for**

---

We now establish the convergence guarantees of PG-BCD+*Lipschitz*. In fact, the development of rigorous statements for the convergence of $\mathbf{B}$, $\mathbf{U}$, $\mathbf{E}$, $\mathbf{V}$, and $\mathbf{M}$ to an optimum is not trivial, due to the block multi-convex nature. Nevertheless, we can establish the convergence of PG-BCD+*Lipschitz* based on a prior analysis of BCD for multiconvex optimization [33]. To achieve this, for the sake of convenience, let $\Theta = (\mathbf{B}, \mathbf{U}, \mathbf{E}, \mathbf{V}, \mathbf{M})$, then we have the following theorem.

**Theorem 2.** *Given any start point $\Theta^{(0)}$, let $\{\Theta^{(l)}\}$ be the sequence of the factors from PG-BCD+Lipschitz, where $l = 1, 2, \cdots$ are the outer iteration numbers, then the sequence $\{\Theta^{(l)}\}$ converges to the finite the critical point of problem (5). In particular, if $\Theta^{(0)}$ is close to the global point of problem (5), PG-BCD+Lipschitz converges to the global optimum.*

Since minimizing AE-NMCF follows multi-block coordinate descent solutions, which correspond to BCDs with the update (1.3a) in [33], we can use the results laid by Xu and Yin [33, Lemma 2.6, Corollary 2.7, and Theorem 2.8] to prove Theorem 2, and the proof details refer to Appendix E. Note that we can not guarantee the global optimum convergence of PG-BCD+*Lipschitz* from an arbitrary point due to the multi-convex, but the use of multiple randomized initialization attempts can increase the change to reach the global optimal solution.

## 4 Experiments

**Data set description.** We use real-world students' response data with different sparsities and knowledge-exercise relations, which are from diversified academic subjects, including $(a)$ *Math* (FrcSub, Junyi-s, and Quanlang-s), $(b)$ *Biology* (SLP-Bio-s), $(c)$ *History* (SLP-His-s), and $(d)$ *English* (SLP-Eng). FrcSub comprises of the fraction subtraction problem scores of 536 middle school students [10]. Junyi-s includes problem logs from an e-learning website based on the open-source code released by Khan Academy [34]. The private Quanlang-s data set is collected from mathematical exams given to junior schools supplied by QUANLANG education company. [2] Others include SLP-

Bio-s, -His-s, and -Eng, which provide unit test results of K-12 learners compiled by an online learning platform (smart learning partner, SLP) [35]. Statistics of the data set are summarized in Table III of Appendix F.

**Baseline approaches.** The baselines include data mining approaches and cognitive diagnosis models. The former uses the well-known **NMF** [36], **MCF** [16], **GNMF** [37], **NMMF** [38], and the advanced **SNMCF** [4]. For the latter, we consider the following competing models: ($i$) **DINA** [10], a classic CDM that models students' knowledge levels by a binary attribute vector with the slip and guess factors of exercises; ($ii$) **DIRT** [11], an extended IRT model incorporating the deep learning technique to enhance the diagnostic process; ($iii$) **DeepCDF** [12], a deep learning-based CDM that considers the importance and relationships of knowledge concepts; and ($iv$) **QRCDM** [13], which integrates the implicit knowledge-exercise relations into CDMs. DIRT and DeepCDF are modified by excluding text information. Our code is available at `https://github.com/ShenbaoYu/AE-NMCF`.

## 4.1 Results

In this section, we evaluate the effectiveness of AE-NMCF in the two learning tasks. Additional experiments, such as cognitive case studies, can be found in Appendix F.

**We first compare student performance prediction.** The evaluation metrics are the commonly used ACC and RMSE [4], which are calculated based on the ground truth of students' responses to exercises and corresponding predicted ones. Table 1 shows the prediction results with the best performances highlighted in boldface, the top 2 results are shaded, and we use '±' to denote the standard deviations. The last column lists the average ranks of all models from the Friedman test (a rank-based method to validate the performance of multiple models on multiple datasets) [39]. In Table 1, we observe that the data mining methods (especially SNMCF and AE-NMCF) perform well, and AE-NMCF lies in the top-2 performers on all data sets except for FrcSub and SLP-Eng in terms of ACC and RMSE, respectively. Its rank of 1.33 on ACC and 1.67 on RMSE also confirm the competitiveness of AE-NMCF. The results indicate that our model can not only handle the students' response data that yields varied degrees of sparsity but also do so for diversified subject domains.

Table 1: Experimental results on student performance prediction

| Metric | Model | Data set | | | | | | Rank |
|---|---|---|---|---|---|---|---|---|
| | | FrcSub | Junyi-s | Quanlang-s | SLP-Bio-s | SLP-His-s | SLP-Eng | |
| ACC ↑ | NMF | 0.7564±0.0093 | 0.6186±0.0223 | 0.6312±0.0075 | 0.6752±0.0103 | 0.7169±0.0094 | 0.7222±0.0114 | 5.83 |
| | MCF-Gra[1] | 0.5727±0.0126 | 0.5046±0.0219 | 0.5679±0.0249 | 0.5515±0.0199 | 0.5828±0.0090 | 0.5640±0.0101 | 10.17 |
| | MCF-New[2] | 0.7066±0.0076 | 0.5327±0.0174 | 0.5743±0.0093 | 0.5543±0.0096 | 0.5824±0.0043 | 0.5696±0.0083 | 9.17 |
| | GNMF | 0.7516±0.0112 | 0.6429±0.0305 | 0.5894±0.0109 | 0.6090±0.0143 | 0.6209±0.0036 | 0.6034±0.0077 | 7.00 |
| | NMMF | 0.7759±0.0085 | 0.6729±0.0264 | 0.6477±0.0194 | 0.6780±0.0084 | 0.7002±0.0041 | 0.7007±0.0176 | 4.83 |
| | SNMCF | 0.8548±0.0043 | 0.6878±0.0192 | 0.7417±0.0051 | 0.7351±0.0108 | 0.8051±0.0032 | 0.7456±0.0091 | 1.83 |
| | DINA | 0.8156±0.0037 | 0.5209±0.0078 | 0.6000±0.0143 | 0.4988±0.0066 | 0.5814±0.0051 | 0.5950±0.0123 | 8.50 |
| | DIRT | 0.6154±0.0076 | 0.5741±0.0208 | 0.6420±0.0100 | 0.5226±0.0222 | 0.5992±0.0150 | 0.5823±0.0181 | 8.33 |
| | DeepCDF | 0.8115±0.0081 | 0.4717±0.0035 | 0.6956±0.0189 | 0.6763±0.0069 | 0.7850±0.0022 | 0.6676±0.0113 | 5.67 |
| | QRCDM | 0.8308±0.0079 | 0.6406±0.0226 | 0.6611±0.0133 | 0.6996±0.0091 | 0.8016±0.0035 | 0.7396±0.0208 | 3.33 |
| | AE-NMCF | 0.8267±0.0048 | **0.7065±0.0285** | **0.7531±0.0064** | **0.7553±0.0101** | **0.8072±0.0019** | **0.7632±0.0062** | 1.33 |
| RMSE ↓ | NMF | 0.4102±0.0057 | 0.5192±0.0105 | 0.4812±0.0052 | 0.4558±0.0099 | 0.4421±0.0068 | 0.4696±0.0093 | 5.67 |
| | MCF-Gra | 0.5677±0.0016 | 0.6762±0.0339 | 0.5393±0.0096 | 0.5487±0.0110 | 0.6621±0.0263 | 0.6675±0.0179 | 10.50 |
| | MCF-New | 0.4738±0.0128 | 0.5906±0.0421 | 0.5602±0.0128 | 0.5674±0.0038 | 0.5897±0.0138 | 0.6478±0.0097 | 9.83 |
| | GNMF | 0.4153±0.0086 | 0.4980±0.0198 | 0.5012±0.0056 | 0.5175±0.0064 | 0.5294±0.0018 | 0.5468±0.0060 | 7.83 |
| | NMMF | 0.3986±0.0032 | 0.4704±0.0402 | 0.4749±0.0076 | 0.4564±0.0083 | 0.4455±0.0062 | 0.4724±0.0226 | 5.33 |
| | SNMCF | **0.3349±0.0029** | 0.4537±0.0189 | 0.4216±0.0076 | 0.4236±0.0082 | 0.3741±0.0069 | 0.5845±0.0867 | 3.17 |
| | DINA | 0.3927±0.0035 | 0.6179±0.0055 | 0.5756±0.0119 | 0.5332±0.0040 | 0.5281±0.0029 | 0.5876±0.0105 | 8.67 |
| | DIRT | 0.4811±0.0014 | 0.4912±0.0075 | 0.4704±0.0037 | 0.4983±0.0023 | 0.4872±0.0011 | 0.4797±0.0040 | 6.83 |
| | DeepCDF | 0.3522±0.0025 | **0.3874±0.0036** | 0.4433±0.0064 | 0.4575±0.0040 | 0.3863±0.0012 | **0.3691±0.0075** | 3.00 |
| | QRCDM | 0.3555±0.0036 | 0.4809±0.0099 | 0.4607±0.0036 | 0.4559±0.0030 | 0.3685±0.0013 | 0.4213±0.0136 | 3.50 |
| | AE-NMCF | 0.3476±0.0088 | 0.4514±0.0193 | 0.4067±0.0047 | **0.3996±0.0060** | **0.3665±0.0024** | 0.4262±0.0034 | 1.67 |

[1] The MCF model with the gradient-based method.
[2] The MCF model with the Newton-Raphson method.

**We proceed to discuss AE-NMCF's ability to estimate students' knowledge proficiency, which is our major concern.** Since the ground truth of students' cognitive levels is unknown, we take cues from the *area under curve* and use a ranking-based metric (*knowledge-response consistency coefficient*, KRC) as an alternative way to evaluate the diagnostic results. Specifically, for knowledge concept $Kc_k$, we first extract the pair set $\mathcal{S} = \{(Ex_n, St_m), n \in [0, N], m \in [0, M]\}$ from the testing set $\mathbf{D}$.

For each pair $(\mathrm{Ex}_n, \mathrm{St}_m)$, we record student $\mathrm{St}_m$'s proficiency level of $\mathrm{Kc}_k$ and the true response score on exercise $\mathrm{Ex}_n$. Then, the KRC result on $\mathrm{Kc}_k$ is $\mathrm{KRC}(\mathrm{Kc}_k) = \left(\chi - \frac{\mathrm{N}^+(\mathrm{N}^++1)}{2}\right)/(\mathrm{N}^+\mathrm{N}^-)$, where $\chi = \sum_{\mathbf{X}^*_{nm}=1} \mathcal{R}(n, m)$, and $\mathcal{R}(n, m)$ is the reordered position of the pair $(\mathrm{Ex}_n, \mathrm{St}_m)$ based on the proficiency level. $\mathrm{N}^+$ ($\mathrm{N}^-$) denotes the number of records with correct (wrong) answers in $\mathcal{S}$. Finally, we average the KRC values of all the knowledge concepts and denote the average as $r_c$. Higher values of $r_c$ indicate better performance.

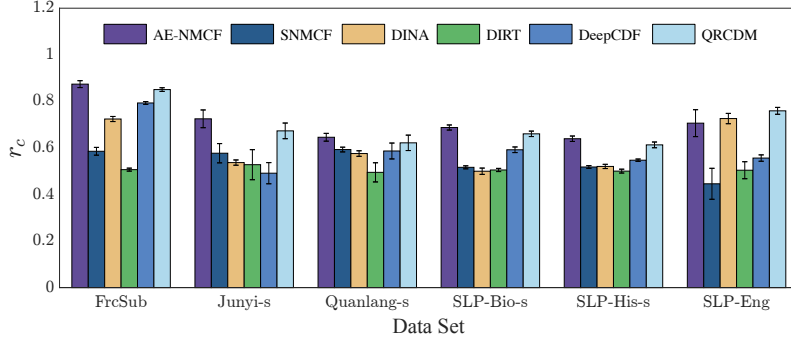

Figure 3: Students' knowledge proficiency estimations.

Figure 3 illustrates the results of estimating students' knowledge proficiency. We exclude NMF, MCF, GNMF, and NMMF due to their known poor capability of cognitive diagnosis. From the results, we have: $(a)$ the $r_c$ values in FrcSub surpass the other data sets as expected, which is mainly due to the strong and consistent connection between exercises and knowledge concepts. $(b)$ Regardless of the relationship types, AE-NMCF delivers comparable or slightly improved performance w.r.t. CDM-based approaches and rises well above SNMCF on all data sets. The resulting $p$-value given by a Wilcoxon-signed rank test between AE-NMCF and SNMCF is 0.031, which also confirms the improvement. $(c)$ While QRCDM shows good diagnostic results, its predicting performance suffers for multiple data sets (see Table 1). This is mainly due to the knowledge-exercise relationship being one-to-one (or one-to-many), which may impede the discovery of implicit correlations.

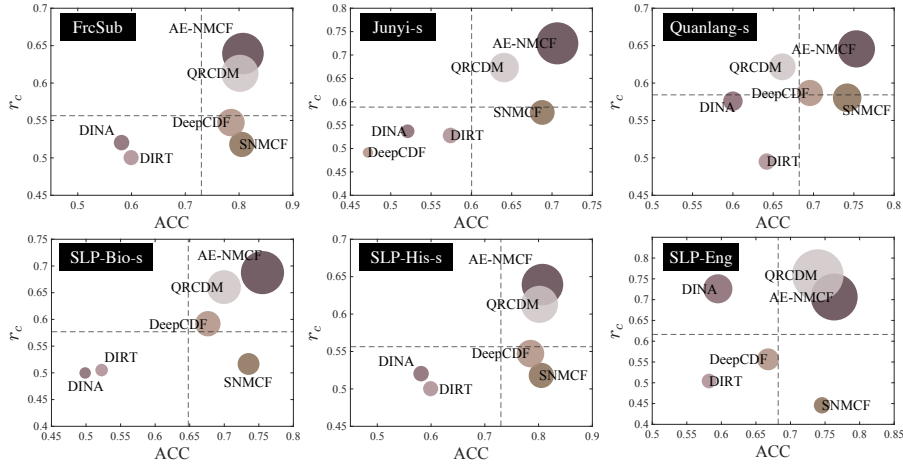

Figure 4: Model comparison in balancing the two learning tasks via bubble visualizations. The x(y)-axis denotes the prediction (estimation) performance in terms of ACC ($r_c$), and the bubble size measures the harmonic mean of ACC and $r_c$. The dash lines locate the models' average performance.

Furthermore, Figure 4 compares the model performance in balancing the two learning tasks, where we use a bubble's horizontal (vertical) position to note the ACC ($r_c$) value for a model. In spired by $F_1$ score, we further visualize the bubble size based on the harmonic mean of ACC and $r_c$. Hence, the closer to the upper right corner with a larger bubble size, the better the balance achieved. As shown in Figure 4, SNMCF excels at student performance prediction but is inadequate in knowledge cognitive estimation. In addition, the comparatively low prediction performance of QRCDM compromises

its balance ability, especially on Junyi-s and Quanlang-s. AE-NMCF, in contrast, is well above the model average (indicated by dash lines) on all data sets, which achieves the best balance between prediction accuracy and diagnostic ability and works with multiple relation cases.

**We close with the demonstration of the effectiveness of our encoder-decoder learning pipeline.** As shown in Table 2, we conduct the ablation study by the use of two variants of AE-NMCF, i.e., AE-NMCF *w/o* Decoder (Encoder) that removes the decoder (encoder) module. The optimization approach is also PG-BCD with appropriate *Lipschitz* constants. According to Table 2, the ignorance of the encoder (or decoder) process leads to a degradation in predicting and estimating performances, and the performance losses of AE-NMCF *w/o* Encoder are lower than those of the variant that removes the decoder. The positive results not only suggest the performance boost of the decoder module but also prove the efficacy of the proposed encoder-decoder architecture, which aligns with our expectations to achieve the monotonicity.

Table 2: Ablation analysis of AE-NMCF in student cognitive modeling

| Metric | Variant | Data set | | | | | |
|---|---|---|---|---|---|---|---|
| | | FrcSub | Junyi-s | Quanlang-s | SLP-Bio-s | SLP-His-s | SLP-Eng |
| ACC ↑ | AE-NMCF *w/o* Decoder | 0.7523±0.0118 | 0.6261±0.0524 | 0.6187±0.0278 | 0.6582±0.0099 | 0.7224±0.0062 | 0.5511±0.0184 |
| | AE-NMCF *w/o* Encoder | 0.8156±0.0060 | 0.6504±0.0141 | 0.7269±0.0066 | 0.7418±0.0040 | 0.7745±0.0011 | 0.7413±0.0126 |
| | AE-NMCF | **0.8267±0.0048** | **0.7065±0.0285** | **0.7531±0.0064** | **0.7553±0.0101** | **0.8072±0.0019** | **0.7632±0.0062** |
| RMSE ↓ | AE-NMCF *w/o* Decoder | 0.4197±0.0042 | 0.4953±0.0141 | 0.4782±0.0081 | 0.4585±0.0041 | 0.4243±0.0025 | 0.5608±0.0135 |
| | AE-NMCF *w/o* Encoder | 0.3668±0.0061 | 0.5076±0.0138 | 0.4262±0.0070 | 0.4160±0.0021 | 0.4193±0.0014 | 0.4494±0.0145 |
| | AE-NMCF | **0.3476±0.0088** | **0.4514±0.0193** | **0.4067±0.0047** | **0.3996±0.0060** | **0.3665±0.0024** | **0.4262±0.0034** |
| $r_c$ ↑ | AE-NMCF *w/o* Decoder | 0.7202±0.0089 | 0.6362±0.0635 | 0.5594±0.0153 | 0.5730±0.0302 | 0.5439±0.0150 | 0.5279±0.0460 |
| | AE-NMCF *w/o* Encoder | 0.8137±0.0140 | 0.6286±0.0828 | 0.5665±0.0563 | 0.5891±0.0171 | 0.5653±0.0091 | 0.4756±0.0225 |
| | AE-NMCF | **0.8738±0.0147** | **0.7249±0.0380** | **0.6456±0.0167** | **0.6875±0.0109** | **0.6393±0.0116** | **0.7063±0.0578** |

## 4.2 Discussion on the Results

We summarize the key findings. First, AE-NMCF improves on competing approaches on two learning tasks across subject domains, data sparsities, and knowledge-exercise relationships. Notably, the better estimation accuracy of knowledge proficiency benefits from the explicit encoding of the knowledge level for each student, which is then iteratively improved by the novel autoencoder machine that guarantees that knowledge proficiency can cumulatively cause success in exercises. Second, our purely data-driven model estimates interpretable factors to pinpoint a student's strengths and weaknesses, which is helpful for decision-making as we may tailor learning resources.

However, AE-NMCF's improved prediction and estimation accuracy over the baselines (SNMCF in particular) comes at a price of higher computational complexity (e.g., see Table VI in Appendix F). Nevertheless, AE-NMCF is well-suited to scale-based tests, which are common scenarios in the real world because students are often evaluated for a small set of knowledge concepts, and the need for confidence statistics is one of the critical factors. In addition, we observe that the diagnostic results for some knowledge concepts tend to be overoptimistic due to ignorance of the prerequisite structure (e.g., see Figure VI in Appendix F), and one part of ongoing work is exploiting the knowledge prerequisite structure for AE-NMCF to attenuate this problem.

## 5 Conclusion

This paper studies student cognitive modeling from a data mining perspective, in which students' knowledge proficiency estimation is our primary concern. To tackle this problem, we propose the AE-NMCF model. Specifically, we root monotonicity in a co-factorization via the carefully crafted encoder-decoder framework. It achieves the assessment of students' knowledge proficiency end-to-end. Considering the nonconvex nature of the objective function with nonnegative constraints, we develop a projected gradient method based on block coordinate descent with *Lipschitz* to facilitate model learning, in which theoretical convergence is guaranteed. Experiments on real-world data sets show that AE-NMCF embraces the merit of satisfactory ability to measure students' knowledge proficiency while retaining good performance prediction accuracy. The future work is two-fold: (1) Considering the learning dependency of knowledge concepts; (2) Investigating other efficient parameter learning methods and exploring their scalability.

## Acknowledgments

SY was supported by the Natural Science Foundation of Xiamen, China (Grant No. 3502Z20227180), YP was supported by the Natural Science Foundation of Guangdong Province, China (Grant No. 2023A1515010869), the Social Science Planning Project of Jiangxi Province, China (Grant No. 19TQ06), and the Provincial Project on Teaching Reform in Higher Education Institutions in Jiangxi Province, China (Grant No. JXJG-18-4-3). PD was supported by a grant from the National Science Foundation of USA (Grant No. IIS-2312657). ML was supported by the Young Top Talent of Young Eagle Program of Fujian Province, China (Grant No. F21E0011202B01). This work was also supported by the National Natural Science Foundation of China (Grant No. 62276168, 62472363, 62176225).

## Footnotes

[2]This data set was made available to us under an agreement with Quanlang education company (https://www.quanlangedu.com) whose terms included informed consent, privacy protection, and fairness.

[3]Since we focus on the computational time in terms of the size of the input data, the complexity of the probit link function is omitted.

[4] The mastery degree of each knowledge concept is normalized to a range of $[0, 1]$.

## References

[1] Yan Zhuang, Qi Liu, GuanHao Zhao, Zhenya Huang, Weizhe Huang, Zachary Pardos, Enhong Chen, Jinze Wu, and Xin Li. A bounded ability estimation for computerized adaptive testing. In *the 37th Annual Conference on Neural Information Processing Systems*, volume 36, 2023.

[2] Shi Dong, Xueyun Tao, Rui Zhong, Zhifeng Wang, Mingzhang Zuo, and Jianwen Sun. Advanced mathematics exercise recommendation based on automatic knowledge extraction and multi-layer knowledge graph. *IEEE Transactions on Learning Technologies*, 17:776–793, 2023.

[3] Louis V. Dibello, L. Roussos, and W. Stout. 31a review of cognitively diagnostic assessment and a summary of psychometric models. *Handbook of Statistics*, 26:979–1030, 2006.

[4] Shenbao Yu, Yifeng Zeng, Yinghui Pan, and Fan Yang. Snmcf: a scalable non-negative matrix co-factorization for student cognitive modeling. *IEEE Transactions on Knowledge and Data Engineering*, pages 1–14, 2023.

[5] Nguyen Thai-Nghe, Lucas Drumond, Tomáš Horváth, Artus Krohn-Grimberghe, Alexandros Nanopoulos, and Lars Schmidt-Thieme. Factorization techniques for predicting student performance. In *Educational Recommender Systems and Technologies: Practices and Challenges*, pages 129–153. IGI Global, 2012.

[6] Andrew S Lan, Andrew E Waters, Christoph Studer, and Richard G Baraniuk. Sparse factor analysis for learning and content analytics. *Journal of Machine Learning Research*, 15(57):1959–2008, 2014.

[7] Paul R Rosenbaum. Testing the conditional independence and monotonicity assumptions of item response theory. *Psychometrika*, 49:425–435, 1984.

[8] Antoine Guisan, Thomas C Edwards Jr, and Trevor Hastie. Generalized linear and generalized additive models in studies of species distributions: setting the scene. *Ecological Modelling*, 157(2-3):89–100, 2002.

[9] Ronald K Hambleton and Linda L Cook. Latent trait models and their use in the analysis of educational test data. *Journal of Educational Measurement*, 14(2):75–96, 1977.

[10] Jimmy De La Torre. Dina model and parameter estimation: a didactic. *Journal of Educational and Behavioral Statistics*, 34(1):115–130, 2009.

[11] Song Cheng, Qi Liu, Enhong Chen, Zai Huang, Zhenya Huang, Yiying Chen, Haiping Ma, and Guoping Hu. Dirt: deep learning enhanced item response theory for cognitive diagnosis. In *Proceedings of the 28th ACM International Conference on Information and Knowledge Management*, pages 2397–2400, 2019.

[12] Lina Gao, Zhongying Zhao, Chao Li, Jianli Zhao, and Qingtian Zeng. Deep cognitive diagnosis model for predicting students' performance. *Future Generation Computer Systems*, 126:252–262, 2022.

[13] Haowen Yang, Tianlong Qi, Jin Li, Longjiang Guo, Meirui Ren, Lichen Zhang, and Xiaoming Wang. A novel quantitative relationship neural network for explainable cognitive diagnosis model. *Knowledge-Based Systems*, 250:109156, 2022.

[14] Michel C Desmarais and Rhouma Naceur. A matrix factorization method for mapping items to skills and for enhancing expert-based q-matrices. In *International Conference on Artificial Intelligence in Education*, pages 441–450, 2013.

[15] Ke Xu, Rujun Liu, Yuan Sun, Keju Zou, Yan Huang, and Xinfang Zhang. Improve the prediction of student performance with hint's assistance based on an efficient non-negative factorization. *IEICE Transactions on Information and Systems*, 100(4):768–775, 2017.

[16] Ajit P Singh and Geoffrey J Gordon. Relational learning via collective matrix factorization. In *Proceedings of the 14th ACM SIGKDD International Conference on Knowledge Discovery and Data Mining*, pages 650–658, 2008.

[17] Farnaz Sedighin, Massoud Babaie-Zadeh, Bertrand Rivet, and Christian Jutten. Multimodal soft nonnegative matrix co-factorization for convolutive source separation. *IEEE Transactions on Signal Processing*, 65(12):3179–3190, 2017.

[18] Adrian Flanagan, Were Oyomno, Alexander Grigorievskiy, Kuan E Tan, Suleiman A Khan, and Muhammad Ammad-Ud-Din. Federated multi-view matrix factorization for personalized recommendations. In *Machine Learning and Knowledge Discovery in Databases: European Conference*, pages 324–347, 2021.

[19] Wen Wen, Wencui Wang, Zhifeng Hao, and Ruichu Cai. Factorizing time-heterogeneous markov transition for temporal recommendation. *Neural Networks*, 159:84–96, 2023.

[20] Panagiotis Symeonidis, Luca Bellinazzi, Chemseddine Berbague, and Markus Zanker. Safe and effective recommendation of drug combinations based on matrix co-factorization. In *2023 IEEE 36th International Symposium on Computer-based Medical Systems*, pages 634–639, 2023.

[21] Bo Yang, Yu Lei, Jiming Liu, and Wenjie Li. Social collaborative filtering by trust. *IEEE Transactions on Pattern Analysis and Machine Intelligence*, 39(8):1633–1647, 2016.

[22] Jiho Yoo and Seungjin Choi. Bayesian matrix co-factorization: variational algorithm and cramér-rao bound. In *Joint European Conference on Machine Learning and Knowledge Discovery in Databases*, pages 537–552, 2011.

[23] Wei Wang, Yan Huang, Yizhou Wang, and Liang Wang. Generalized autoencoder: a neural network framework for dimensionality reduction. In *Proceedings of the IEEE Conference on Computer Vision and Pattern Recognition Workshops*, pages 490–497, 2014.

[24] Pengfei Liu, Xiaoming Sun, Yang Han, Zhishuai He, Weifeng Zhang, and Chenxu Wu. Arrhythmia classification of lstm autoencoder based on time series anomaly detection. *Biomedical Signal Processing and Control*, 71:103228, 2022.

[25] Dong Gong, Lingqiao Liu, Vuong Le, Budhaditya Saha, Moussa Reda Mansour, Svetha Venkatesh, and Anton van den Hengel. Memorizing normality to detect anomaly: memory-augmented deep autoencoder for unsupervised anomaly detection. In *Proceedings of the IEEE/CVF International Conference on Computer Vision*, pages 1705–1714, 2019.

[26] Viet-Tuan Le and Yong-Guk Kim. Attention-based residual autoencoder for video anomaly detection. *Applied Intelligence*, 53(3):3240–3254, 2023.

[27] Michel C Desmarais. Mapping question items to skills with non-negative matrix factorization. *ACM SIGKDD Explorations Newsletter*, 13(2):30–36, 2012.

[28] Eric R Ziegel. The elements of statistical learning. *Technometrics*, 45(3):267–268, 2003.

[29] Prateek Jain and Purushottam Kar. Non-convex optimization for machine learning. *Foundations and Trends in Machine Learning*, 10(3-4):142–363, 2017.

[30] Chih-Jen Lin. Projected gradient methods for nonnegative matrix factorization. *Neural Computation*, 19(10):2756–2779, 2007.

[31] Dimitri P Bertsekas. Nonlinear programming. *Journal of the Operational Research Society*, 48(3):334–334, 1997.

[32] Juha Heinonen. *Lectures on Lipschitz Analysis*. University of Jyväskylä, 2005.

[33] Yangyang Xu and Wotao Yin. A block coordinate descent method for regularized multiconvex optimization with applications to nonnegative tensor factorization and completion. *SIAM Journal on Imaging Sciences*, 6(3):1758–1789, 2013.

[34] Haw-Shiuan Chang, Hwai-Jung Hsu, and Kuan-Ta Chen. Modeling exercise relationships in e-learning: a unified approach. In *International Conference on Educational Data Mining*, pages 532–535, 2015.

[35] Y Lu, Yang Pian, Ziding Shen, et al. Slp: a multi-dimensional and consecutive dataset from k-12 education. In *Proceedings of the 29th International Conference on Computers in Education Conference*, pages 261–266, 2021.

[36] Daniel D. Lee and H. Sebastian Seung. Algorithms for non-negative matrix factorization. In *International Conference on Neural Information Processing Systems*, pages 556–562, 2000.

[37] Hyekyoung Lee and Seungjin Choi. Group nonnegative matrix factorization for eeg classification. In *Artificial Intelligence and Statistics*, pages 320–327. PMLR, 2009.

[38] Koh Takeuchi, Katsuhiko Ishiguro, Akisato Kimura, and Hiroshi Sawada. Non-negative multiple matrix factorization. In *Proceedings of the 23rd International Joint Conference on Artificial Intelligence*, pages 1713–1720, 2013.

[39] Janez Demšar. Statistical comparisons of classifiers over multiple data sets. *Journal of Machine Learning Research*, 7:1–30, 2006.

[40] Edwin F Beckenbach and Richard Bellman. *Inequalities*, volume 30. Springer Science & Business Media, 2012.

[41] Roger A Horn, Roger A Horn, and Charles R Johnson. *Topics in matrix analysis*. Cambridge University Press, 1994.

[42] Krzysztof Kurdyka. On gradients of functions definable in o-minimal structures. In *Annales de l'institut Fourier*, volume 48, pages 769–783, 1998.

[43] Steven G Krantz and Harold R Parks. *A primer of real analytic functions*. Springer Science & Business Media, 2002.

# Appendix

## A  An Example of the Encoder-Decoder Process

Figure I provides an example of the encoder (**Left**) and decoder (**Right**) processes of AE-NMCF. First, the encoder targets the specification of student $St_3$'s knowledge proficiency vector ($\mathbf{A}_{:3}$). Here, we omit matrix $\mathbf{B}$ for conciseness. Second, the decoder reconstructs $St_3$'s response on exercise $Ex_3$ (i.e., $\mathbf{X}_{33}$) via the specified knowledge proficiency. Thus, by explicitly encoding the knowledge level for each student in the encoder, which is then iteratively improved by the decoder that guarantees that knowledge proficiency can cumulatively cause success in related exercises, monotonicity can be achieved.

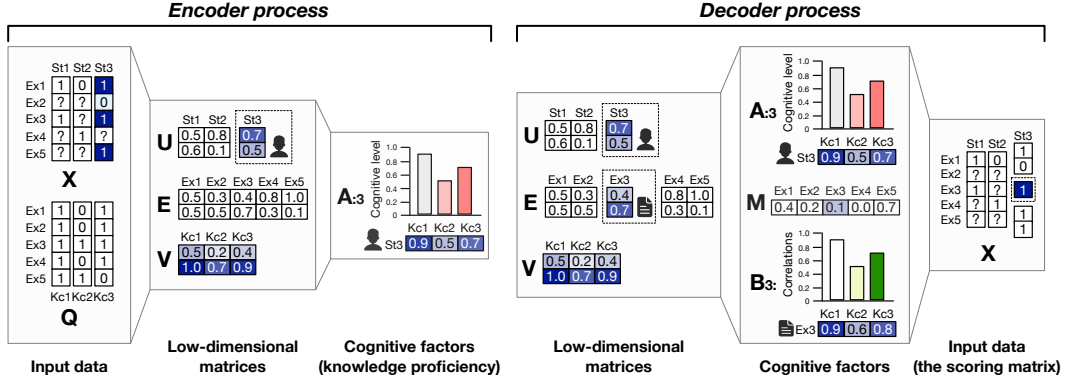

Figure I: An illustration of the encoder and decoder processes, where we highlight the target entries in the color cells.

## B  Parameter Learning for AE-NMCF

The parameters of AE-NMCF include $\mathbf{B}$, $\mathbf{U}$, $\mathbf{E}$, $\mathbf{V}$, and $\mathbf{M}$, where $\mathbf{B} = [\mathbf{B}_{1:}^\top, \mathbf{B}_{2:}^\top, \cdots, \mathbf{B}_{N:}^\top]^\top$, $\mathbf{U} = [\mathbf{U}_{:1}, \mathbf{U}_{:2}, \cdots, \mathbf{U}_{:M}]$, and $\mathbf{M} = [\mu_1, \mu_2, \cdots, \mu_N]^\top$. Because of the nonnegative constraints on $\mathbf{B}$, $\mathbf{U}$, $\mathbf{E}$, and $\mathbf{V}$, we employ the projected gradient via a block coordinate descent (PG-BCD) for the parameters solution, and the subproblems of $\mathbf{B}_{n:}$, $\mathbf{U}_{:m}$, $\mathbf{E}$, $\mathbf{V}$, and $\mu_n$ are

$$\min_{\mathbf{B}_{n:}\geq\mathbf{0}} \ \mathcal{O}_{\text{AF}}(\mathbf{B}_{n:}) = \sum_m -\log \Pr(\mathbf{X}_{nm}) + \frac{\gamma}{2}\|\mathbf{B}_{n:}\|_2^2 + \|\mathbf{Q}_{n:} \odot (\mathbf{B}_{n:} - \mathbf{E}_{n:}\mathbf{V})\|_2^2, \tag{a}$$

$$\min_{\mathbf{U}_{:m}\geq\mathbf{0}} \ \mathcal{O}_{\text{AF}}(\mathbf{U}_{:m}) = \sum_n -\log \Pr(\mathbf{X}_{nm}) + \|\mathbf{W}_{:m} \odot (\mathbf{X}_{:m} - \mathbf{E}\mathbf{U}_{:m})\|_2^2, \tag{b}$$

$$\min_{\mathbf{E}\geq\mathbf{0}} \ \ \mathcal{O}_{\text{AF}}(\mathbf{E}) = \|\mathbf{W} \odot (\mathbf{X} - \mathbf{E}\mathbf{U})\|_{\text{F}}^2 + \|\mathbf{Q} \odot (\mathbf{B} - \mathbf{E}\mathbf{V})\|_{\text{F}}^2, \tag{c}$$

$$\min_{\mathbf{V}\geq\mathbf{0}} \ \ \mathcal{O}_{\text{AF}}(\mathbf{V}) = \sum_{n,m} -\log \Pr(\mathbf{X}_{nm}) + \|\mathbf{Q} \odot (\mathbf{B} - \mathbf{E}\mathbf{V})\|_{\text{F}}^2, \tag{d}$$

$$\min_{\mu_n} \ \ \mathcal{O}_{\text{AF}}(\mu_n) = \sum_m -\log \Pr(\mathbf{X}_{nm}), \tag{e}$$

where $\Pr(\mathbf{X}_{nm}) = \Phi(\Delta_{nm})^{\mathbf{X}_{nm}} [1 - \Phi(\Delta_{nm})]^{(1-\mathbf{X}_{nm})}$, and $\Delta_{nm} = \mathbf{B}_{n:}\mathbf{A}_{:m} + \mu_n$. Considering the computation burden of the Hessian and the calculation difficulty for the probit link function when employing second-order approaches, we build our parameter learning algorithm on first-order methods. Hence, the gradients for problems (a) – (e) are

$$\nabla \mathcal{O}_{\text{AF}}(\mathbf{B}_{n:}) = -\sum_m \Xi_{nm}[\mathbf{X}_{nm} - \Phi(\Delta_{nm})]\mathbf{U}_{:m}^\top \mathbf{V} + 2[\mathbf{Q}_{n:}\odot\mathbf{B}_{n:} - \mathbf{Q}_{n:}\odot(\mathbf{E}_{n:}\mathbf{V})] + \gamma\mathbf{B}_{n:},$$

$$\nabla \mathcal{O}_{\mathrm{AF}}(\mathbf{U}_{:m}) = -\sum_n \Xi_{nm}[\mathbf{X}_{nm} - \Phi(\Delta_{nm})]\mathbf{V}\mathbf{B}_{n:}^\top + 2\mathbf{E}^\top[\mathbf{W}_{:m}\odot(\mathbf{E}\mathbf{U}_{:m}) - \mathbf{W}_{:m}\odot\mathbf{X}_{:m}],$$

$$\nabla \mathcal{O}_{\mathrm{AF}}(\mathbf{E}) = 2[\mathbf{W}\odot(\mathbf{E}\mathbf{U}) - \mathbf{W}\odot\mathbf{X}]\mathbf{U}^\top + 2[\mathbf{Q}\odot(\mathbf{E}\mathbf{V}) - \mathbf{Q}\odot\mathbf{B}]\mathbf{V}^\top,$$

$$\nabla \mathcal{O}_{\mathrm{AF}}(\mathbf{V}) = -\sum_{(n,m)} \Xi_{nm}[\mathbf{X}_{nm} - \Phi(\Delta_{nm})]\mathbf{U}_{:m}\mathbf{B}_{n:} + 2\mathbf{E}^\top[\mathbf{Q}\odot(\mathbf{E}\mathbf{V}) - \mathbf{Q}\odot\mathbf{B}],$$

$$\nabla \mathcal{O}_{\mathrm{AF}}(\mu_n) = -\sum_m \Xi_{nm}[\mathbf{X}_{nm} - \Phi(\Delta_{nm})],$$

where $\Xi_{nm} = \frac{\mathcal{N}(\Delta_{nm})}{\Phi(\Delta_{nm})[1-\Phi(\Delta_{nm})]}$. Based on the gradients above, searching for the optimum point is easy. Taking $\mathbf{B}_{n:}$ as an example, in each iteration $l = 1, 2, \cdots$, the gradient step is

$$\mathbf{B}_{n:}^{(l+1)} \leftarrow \left[\mathbf{B}_{n:}^{(l)} - \eta_{\mathbf{B}_{n:}}^{(l)}\nabla\mathcal{O}_{\mathrm{AF}}^{(l)}(\mathbf{B}_{n:})\right]_+,$$

where we use a half-wave rectifier $[x]_+ = max(\epsilon, x)$, $\epsilon = 10^{-15}$ to ensure the nonnegativity, and $\eta_{\mathbf{B}_{n:}}^{(l)}$ is a suitable step size for $\mathbf{B}_{n:}$, which is determined by the *Lipschitz* constant in this paper.

## C    Complexity Analysis

This section discusses the time complexity of PG-BCD+*Lipschitz*. The analysis is based on the update rules of $\mathbf{B}_{n:}$, $\mathbf{U}_{:m}$, $\mathbf{E}$, $\mathbf{V}$, and $\mu_{:m}$ ($1\leq n\leq$N, $1\leq m\leq$M). For simplicity of exposition, we consider the case where the number of students is larger than that of knowledge concepts, which commonly occurs. For each block $\mathbf{B}_{n:}$, the *Lipschitz* constant takes $O(\mathrm{MK}^2)$ operations, and the operations of the gradient are bounded by $O(\mathrm{MKT})$ for each iteration.[3] Consequently, the cost of the variable $\mathbf{B}$, which contains N blocks, is bounded by $O(\mathrm{MNK}^2)$. Other parameters can be analyzed similarly, summarized in Table I. Hence, the overall cost is the number of iterations needed for convergence times $O(\mathrm{MNK}^2)$, where the latter term is the complexity of each iteration.

Table I: Computational operations $(\mathbf{U}, \mathbf{E}, \mathbf{V}, \mathbf{M})$ for each iteration

| U | E | V | M |
|---|---|---|---|
| $O(\mathrm{MNKT})$ | $O(min(\mathrm{M, N})\cdot\mathrm{MN})$ | $O(max(\mathrm{KT}, min(\mathrm{M, N}))\cdot\mathrm{MN})$ | $O(\mathrm{MNKT})$ |

## D    Proof of Theorem 1

To prove Theorem 1, we first introduce a scalar *Lipschitz* constant in Lemma i [6].

**Lemma i.** *Let* $g(x) = \frac{\Phi'(x)}{\Phi(x)}$, $x \in \mathbb{R}$, *where* $\Phi(x)$ *is the inverse probit function. Then, for* $y, z \in \mathbb{R}$, *we have*

$$|g(y) - g(x)| \leq L_p|y - z|,$$

*where* $L_p = 1$ *is the scalar Lipschitz constant for* $g(x)$.

Next, we prove Theorem 1. For the sake of brevity, we assume that all entries in the student scoring matrix $\mathbf{X}$ are observed, i.e., $\Omega_o = \{1, \cdots, N\}\times\{1, \cdots, M\}$; the extension to the case with missing entries in $\mathbf{X}$ is straightforward. In what follows, $\mathcal{N}(\cdot)$ and $\Phi(\cdot)$ are assumed to operate element-wise on the vector or matrix. We start with the gradient of $\mathcal{O}_{\mathrm{AF}}(\mathbf{B}_{n:})$ in Theorem 1, as shown below

$$\nabla\mathcal{O}_{\mathrm{AF}}(\mathbf{B}_{n:}) = -\underbrace{\sum_{m=1}^{\mathrm{M}}\left\{\frac{\mathcal{N}(\Lambda_{nm})}{\Phi(\Lambda_{nm})}(2\mathbf{X}_{nm} - 1)\mathbf{U}_{:m}^\top\mathbf{V}\right\}}_{\mathcal{O}_{\mathrm{AF}}^{(1)}(\mathbf{B}_{n:})} + 2[\mathbf{Q}_{n:}\odot\mathbf{B}_{n:} - \mathbf{Q}_{n:}\odot(\mathbf{E}_{n:}\mathbf{V})] + \gamma\mathbf{B}_{n:},$$

where $\Lambda_{nm} = (2\mathbf{X}_{nm} - 1)(\mathbf{B}_{n:}\mathbf{V}^\top\mathbf{U}_{:m} + \mu_n)$. The first term $\mathcal{O}_{\mathrm{AF}}^{(1)}(\mathbf{B}_{n:})$ can be rearranged as

$$\nabla\mathcal{O}_{\mathrm{AF}}^{(1)}(\mathbf{B}_{n:}) = \frac{\mathcal{N}(\Lambda_{n1})}{\Phi(\Lambda_{n1})}(2\mathbf{X}_{n1} - 1)\mathbf{U}_{:1}^\top\mathbf{V} + \cdots + \frac{\mathcal{N}(\Lambda_{n\mathrm{M}})}{\Phi(\Lambda_{n\mathrm{M}})}(2\mathbf{X}_{n\mathrm{M}} - 1)\mathbf{U}_{:\mathrm{M}}^\top\mathbf{V}$$

$$= \left[\frac{\mathcal{N}(\Lambda_{n1})}{\Phi(\Lambda_{n1})}, \cdots, \frac{\mathcal{N}(\Lambda_{n\mathrm{M}})}{\Phi(\Lambda_{n\mathrm{M}})}\right]\begin{bmatrix}(2\mathbf{X}_{n1} - 1)\mathbf{U}_{:1}^\top\mathbf{V} \\ \vdots \\ (2\mathbf{X}_{n\mathrm{M}} - 1)\mathbf{U}_{:\mathrm{M}}^\top\mathbf{V}\end{bmatrix}$$

$$= \frac{\mathcal{N}(\Lambda_{n:})}{\Phi(\Lambda_{n:})}\widetilde{\mathbf{C}}_n,$$

where $\Lambda_{n:} = [\Lambda_{n1}, \cdots, \Lambda_{n\mathrm{M}}]$, and we have

$$\begin{aligned}\Lambda_{n:} &= [(2\mathbf{X}_{n1} - 1)(\mathbf{B}_{n:}\mathbf{V}^\top\mathbf{U}_{:1} + \mu_n), \cdots, (2\mathbf{X}_{n\mathrm{M}} - 1)(\mathbf{B}_{n:}\mathbf{V}^\top\mathbf{U}_{:\mathrm{M}} + \mu_n)] \\ &= \mathbf{B}_{n:}[(2\mathbf{X}_{n1} - 1)\mathbf{V}^\top\mathbf{U}_{:1}, \cdots, (2\mathbf{X}_{n\mathrm{M}} - 1)\mathbf{V}^\top\mathbf{U}_{:\mathrm{M}}] + [(2\mathbf{X}_{n1} - 1)\mu_n, \cdots, (2\mathbf{X}_{n\mathrm{M}} - 1)\mu_n] \\ &= \mathbf{B}_{n:}\widetilde{\mathbf{C}}_n^\top + \widetilde{\mathbf{X}}_{n:}^{\mu_n}.\end{aligned}$$

Therefore, the gradient $\nabla\mathcal{O}_{\mathrm{AF}}(\mathbf{B}_{n:})$ can be rewritten as

$$\nabla\mathcal{O}_{\mathrm{AF}}(\mathbf{B}_{n:}) = -\frac{\mathcal{N}(\mathbf{B}_{n:}\widetilde{\mathbf{C}}_n^\top + \widetilde{\mathbf{X}}_{n:}^{\mu_n})}{\Phi(\mathbf{B}_{n:}\widetilde{\mathbf{C}}_n^\top + \widetilde{\mathbf{X}}_{n:}^{\mu_n})}\widetilde{\mathbf{C}}_n + 2[\mathbf{Q}_{n:}\odot\mathbf{B}_{n:} - \mathbf{Q}_{n:}\odot(\mathbf{E}_{n:}\mathbf{V})] + \gamma\mathbf{B}_{n:}.$$

We can now establish an upper bound of the $l_2$-norm of the difference between the gradients at two arbitrary points $\mathbf{y}$ and $\mathbf{z}$ of $\nabla\mathcal{O}_{\mathrm{AF}}(\mathbf{B}_{n:})$ as follows

$$\|\nabla\mathcal{O}_{\mathrm{AF}}(\mathbf{y}) - \nabla\mathcal{O}_{\mathrm{AF}}(\mathbf{z})\|_2$$

$$= \left\|-\frac{\mathcal{N}(\mathbf{y}\widetilde{\mathbf{C}}_n^\top + \widetilde{\mathbf{X}}_{n:}^{\mu_n})}{\Phi(\mathbf{y}\widetilde{\mathbf{C}}_n^\top + \widetilde{\mathbf{X}}_{n:}^{\mu_n})}\widetilde{\mathbf{C}}_n + 2[\mathbf{Q}_{n:}\odot\mathbf{y} - \mathbf{Q}_{n:}\odot(\mathbf{E}_{n:}\mathbf{V})]\right.$$

$$\left.+\frac{\mathcal{N}(\mathbf{z}\widetilde{\mathbf{C}}_n^\top + \widetilde{\mathbf{X}}_{n:}^{\mu_n})}{\Phi(\mathbf{z}\widetilde{\mathbf{C}}_n^\top + \widetilde{\mathbf{X}}_{n:}^{\mu_n})}\widetilde{\mathbf{C}}_n - 2[\mathbf{Q}_{n:}\odot\mathbf{z} - \mathbf{Q}_{n:}\odot(\mathbf{E}_{n:}\mathbf{V})] + \gamma\mathbf{y} - \gamma\mathbf{z}\right\|_2$$

$$\leq |-1|\left\|\frac{\mathcal{N}(\mathbf{y}\widetilde{\mathbf{C}}_n^\top + \widetilde{\mathbf{X}}_{n:}^{\mu_n})}{\Phi(\mathbf{y}\widetilde{\mathbf{C}}_n^\top + \widetilde{\mathbf{X}}_{n:}^{\mu_n})}\widetilde{\mathbf{C}}_n - \frac{\mathcal{N}(\mathbf{z}\widetilde{\mathbf{C}}_n^\top + \widetilde{\mathbf{X}}_{n:}^{\mu_n})}{\Phi(\mathbf{z}\widetilde{\mathbf{C}}_n^\top + \widetilde{\mathbf{X}}_{n:}^{\mu_n})}\widetilde{\mathbf{C}}_n\right\|_2$$

$$+ \|2[\mathbf{Q}_{n:}\odot\mathbf{y} - \mathbf{Q}_{n:}\odot(\mathbf{E}_{n:}\mathbf{V})] - 2[\mathbf{Q}_{n:}\odot\mathbf{z} - \mathbf{Q}_{n:}\odot(\mathbf{E}_{n:}\mathbf{V})]\|_2 + \gamma\|\mathbf{y} - \mathbf{z}\|_2 \tag{f}$$

$$\leq \sigma_1(\widetilde{\mathbf{C}}_n)\left\|\frac{\mathcal{N}(\mathbf{y}\widetilde{\mathbf{C}}_n^\top + \widetilde{\mathbf{X}}_{n:}^{\mu_n})}{\Phi(\mathbf{y}\widetilde{\mathbf{C}}_n^\top + \widetilde{\mathbf{X}}_{n:}^{\mu_n})} - \frac{\mathcal{N}(\mathbf{z}\widetilde{\mathbf{C}}_n^\top + \widetilde{\mathbf{X}}_{n:}^{\mu_n})}{\Phi(\mathbf{z}\widetilde{\mathbf{C}}_n^\top + \widetilde{\mathbf{X}}_{n:}^{\mu_n})}\right\|_2$$

$$+ 2\|\mathbf{Q}_{n:}\odot\mathbf{y} - \mathbf{Q}_{n:}\odot\mathbf{z}\|_2 + \gamma\|\mathbf{y} - \mathbf{z}\|_2 \tag{g}$$

$$\leq \sigma_1(\widetilde{\mathbf{C}}_n)L_p\left\|(\mathbf{y}\widetilde{\mathbf{C}}_n^\top + \widetilde{\mathbf{X}}_{n:}^{\mu_n}) - (\mathbf{z}\widetilde{\mathbf{C}}_n^\top + \widetilde{\mathbf{X}}_{n:}^{\mu_n})\right\|_2 + 2\|\mathbf{Q}_{n:}\|_2\|\mathbf{y} - \mathbf{z}\|_2 + \gamma\|\mathbf{y} - \mathbf{z}\|_2 \tag{h}$$

$$\leq L_p\sigma_1^2(\widetilde{\mathbf{C}}_n)\|\mathbf{y} - \mathbf{z}\|_2 + 2\|\mathbf{Q}_{n:}\|_2\|\mathbf{y} - \mathbf{z}\|_2 + \gamma\|\mathbf{y} - \mathbf{z}\|_2 \tag{i}$$

$$= \left[L_p\sigma_1^2(\mathbf{U}^\top\mathbf{V}) + 2\left(\sum_{k=1}^{\mathrm{K}}\mathbf{Q}_{nk}^2\right)^{\frac{1}{2}} + \gamma\right]\|\mathbf{y} - \mathbf{z}\|_2.$$

Here, (f) uses the triangle inequality of a norm. (g) follows the Hölder inequality [40], and $\sigma_1(\cdot)$ denotes the corresponding maximum singular value. The bounds of (h) follow from Lemma i and the inequality of Hadamard products (e.g., see [41, Section 5.5.1]). The final inequality (i) follows from the fact that flipping the signs of the rows (or columns) of a matrix does not affect its singular values.

Note that this proof assumes that the scoring matrix $\mathbf{X}$ is fully populated. We can easily adapt to the case of missing entries in $\mathbf{X}$, by replacing the matrix $\widetilde{\mathbf{C}}_n$ to $\widetilde{\mathbf{C}}_n^{\mathcal{I}}$, which is the matrix containing the

rows of $\widetilde{\mathbf{C}}_n$ corresponding to the observed entries indexed by the set $\mathcal{I} = \{m : (n, m) \in \Omega_o\}$. We omit the details for the sake of brevity.

The proofs for the remaining subproblems for $\mathbf{U}_{:m}$, $\mathbf{E}$, $\mathbf{V}$, and $\mu_n$ ($1 \leq m \leq M, 1 \leq n \leq N$) follow analogously, and Table II summarizes the *Lipschitz* constants of all parameters, where $\mathbf{1}_{M,N}$ denotes the $M \times N$ all-ones matrix.

<div align="center">

Table II: The *Lipschitz* constants of parameters

| Parameter | The *Lipschitz* constant |
|---|---|
| $\mathbf{B}_{n:}$ | $L_p \sigma_1^2(\mathbf{U}^\top \mathbf{V}) + 2 \left( \sum_{k=1}^{K} \mathbf{Q}_{nk}^2 \right)^{\frac{1}{2}} + \gamma$ |
| $\mathbf{U}_{:m}$ | $L_p \sigma_1^2(\mathbf{V}\mathbf{B}^\top) + 2\sigma_1^2(\mathbf{E}) \left( \sum_{n=1}^{N} \mathbf{W}_{nm}^2 \right)^{\frac{1}{2}}$ |
| $\mathbf{E}$ | $2\sigma_1^2(\mathbf{U})\sigma_1(\mathbf{W}) + 2\sigma_1^2(\mathbf{V})\sigma_1(\mathbf{Q})$ |
| $\mathbf{V}$ | $L_p \sigma_1^2(\mathbf{U})\sigma_1^2(\mathbf{B})\sigma_1^2(2\mathbf{X}^\top - \mathbf{1}_{M,N}) + 2\sigma_1^2(\mathbf{E})\sigma_1(\mathbf{Q})$ |
| $\mu_n$ | $L_p \sum_{m=1}^{M} (2\mathbf{X}_{nm} - 1)^2$ |

</div>

## E  Proof of Theorem 2

Since minimizing AE-NMCF follows the multi-block coordinate descent solution, and the subproblems can correspond to BCDs with update (1.3a) in [33], we use the results laid by Xu and Yin [33, Lemma 2.6, Corollary 2.7, and Theorem 2.8] to prove the convergence of PG-BCD+*Lipschitz*. To this end, we show that the objective function of AE-NMCF, i.e., problem (j) meets all assumptions needed for the convergence results in [33].

$$\min_{\mathbf{B},\mathbf{U},\mathbf{E},\mathbf{V},\mathbf{M}} \quad \mathcal{O}_{\mathrm{AF}} = -\ell + \|\mathbf{W} \odot (\mathbf{X} - \mathbf{E}\mathbf{U})\|_{\mathrm{F}}^2 + \|\mathbf{Q} \odot (\mathbf{B} - \mathbf{E}\mathbf{V})\|_{\mathrm{F}}^2 + \frac{\gamma}{2}\sum_{n=1}^{N} \|\mathbf{B}_{n:}\|_2^2, \qquad \text{(j)}$$

$$\text{s.t.} \quad \mathbf{B} \geq \mathbf{0}, \mathbf{U} \geq \mathbf{0}, \mathbf{E} \geq \mathbf{0}, \mathbf{V} \geq \mathbf{0}.$$

We start by discussing Assumptions 1 and 2 in [33]. For Assumption 1, since all the terms in problem (j) are nonnegative, we have $\mathcal{O}_{\mathrm{AF}} > -\infty$, which has a lower bound of 0. For Assumption 2, by inspecting the form of the individual subproblems, we see that they are strongly convex. Therefore, Assumptions 1 and 2 in [33] are met.

We then provide that problem (j) also meets the additional assumptions in [33, Lemma 2.6], which requires $(a)$ the *Lipschitz* continuous of the gradient of the block multi-convex function $\nabla f$ on any bound set and $(b)$ the Kurdyka-Łojasiewicz (KL) inequality [42]. To do so, for $(a)$, let $\Theta = (\mathbf{B}, \mathbf{U}, \mathbf{E}, \mathbf{V}, \mathbf{M})$, and we can rewrite the objective function of problem (j) as follows

$$\mathcal{O}_{\mathrm{AF}}(\Theta) = -\ell + \|\mathbf{W} \odot (\mathbf{X} - \mathbf{E}\mathbf{U})\|_{\mathrm{F}}^2 + \|\mathbf{Q} \odot (\mathbf{B} - \mathbf{E}\mathbf{V})\|_{\mathrm{F}}^2 + \frac{\gamma}{2}\sum_{n=1}^{N} \|\mathbf{B}_{n:}\|_2^2$$

$$+ \sum_{\Theta_{(i)} \in \Theta} \delta(\Theta_{(i)} < \mathbf{0})$$

$$= \mathcal{O}_{\mathrm{AF}}^s(\Theta) + \sum_{\Theta_{(i)} \in \Theta} \delta(\Theta_{(i)} < \mathbf{0}),$$

where $\Theta_{(i)}$ denotes the $i$-th element of $\Theta$. $\delta(z)$ is an indicator function, and we have $\delta(z) = \infty$ if $z < 0$ and 0 otherwise. We now show that the gradients of the smooth part of $\mathcal{O}_{\mathrm{AF}}(\Theta)$, i.e., $\nabla\mathcal{O}_{\mathrm{AF}}^s(\Theta)$, is *Lipschitz* continuous in $\mathrm{dom}(\mathcal{O}_{\mathrm{AF}}^s)$.

Let $\Theta^y, \Theta^z \in \mathrm{dom}(\mathcal{O}_{\mathrm{AF}}^s)$, we have

$$\|\nabla\mathcal{O}_{\mathrm{AF}}^s(\Theta^y) - \nabla\mathcal{O}_{\mathrm{AF}}^s(\Theta^z)\|_2 = \left\{ \sum_{\Theta_{(i)} \in \Theta} \left( \nabla\mathcal{O}_{\mathrm{AF}}^s(\Theta_{(i)}^y) - \nabla\mathcal{O}_{\mathrm{AF}}^s(\Theta_{(i)}^z) \right)^2 \right\}^{\frac{1}{2}}$$

$$\leq \left\{ \sum_{\Theta_{(i)} \in \Theta} L_{\Theta_{(i)}}^2 \|\Theta_{(i)}^y - \Theta_{(i)}^z\|_2^2 \right\}^{\frac{1}{2}}$$

$$\leq (L')^{\frac{1}{2}} \|\Theta^y - \Theta^z\|_2,$$

where $L' = \max\{L_{\Theta_{(i)}}^2\}$, $\Theta_{(i)} \in \{\mathbf{B}, \mathbf{U}, \mathbf{E}, \mathbf{V}, \mathbf{M}\}$. Recall that the bounds on the *Lipschitz* constant corresponding to $\mathbf{E}$ and $\mathbf{V}$ are shown in Table II. For $\mathbf{B}$, let $\bar{\mathbf{B}} = [\mathbf{B}_{1:}, \mathbf{B}_{2:}, \cdots, \mathbf{B}_{\mathrm{N}:}]^\top$, we have

$$\|\nabla\mathcal{O}_{\mathrm{AF}}^s(\bar{\mathbf{B}}^y) - \nabla\mathcal{O}_{\mathrm{AF}}^s(\bar{\mathbf{B}}^z)\|_2 = \left\{ \sum_{n=1}^{\mathrm{N}} \left( \nabla\mathcal{O}_{\mathrm{AF}}^s(\mathbf{B}_{n:}^y) - \nabla\mathcal{O}_{\mathrm{AF}}^s(\mathbf{B}_{n:}^z) \right)^2 \right\}^{\frac{1}{2}}$$

$$\leq \left\{ \sum_{n=1}^{\mathrm{N}} \left[ L_p \sigma_1^2(\mathbf{U}^\top \mathbf{V}) + 2 \left( \sum_{k=1}^{\mathrm{K}} \mathbf{Q}_{nk}^2 \right)^{\frac{1}{2}} + \gamma \right]^2 \|\mathbf{B}_{n:}^y - \mathbf{B}_{n:}^z\|_2^2 \right\}^{\frac{1}{2}}$$

$$\leq \left( L_p \|\mathbf{U}^\top \mathbf{V}\|_{\mathrm{F}}^2 + 2\|\mathbf{Q}_{n:}\|_2 + \gamma \right) \|\bar{\mathbf{B}}^y - \bar{\mathbf{B}}^z\|_2,$$

where the last line states that the maximum singular value of a matrix is no greater than its Frobenius norm. Similarly, the *Lipschitz* constants for $\mathbf{U}$ and $\mathbf{M}$ are $L_p\|\mathbf{V}\mathbf{B}^\top\|_{\mathrm{F}}^2 + 2\|\mathbf{E}\|_2^2\|\mathbf{W}_{:m}\|_2$ and $L_p\|2\mathbf{X}_{n:} - \mathbf{1}_{1,\mathrm{M}}\|_2^2$, respectively. Therefore, $\nabla\mathcal{O}_{\mathrm{AF}}^s(\Theta)$ is *Lipschitz* continuous in $\mathrm{dom}(\mathcal{O}_{\mathrm{AF}}^s)$.

For $(b)$, using [6, Lemma 7], the first term (i.e., the negative log-probit likelihood function $-\ell$) of $\mathcal{O}_{\mathrm{AF}}$ in problem (j) is real analytic, which is based on the fact that compositions of real analytic functions are real analytic [43]. In addition, the second and third terms with Frobenius norms in $\mathcal{O}_{\mathrm{AF}}$, plus the regularizer, are all polynomial functions, therefore also real analytic. Hence, the objective function $\mathcal{O}_{\mathrm{AF}}$ is real analytic and satisfies the KL inequality, a consequence of [33, Section 2.2]. By setting the extrapolation weight $\omega_i^k = 0$ in [33], we can conclude that the PG-BCD+*Lipschitz* algorithm converges to a local minimum. Furthermore, PG-BCD+*Lipschitz* converges globally if the initial point is close to the global minimum [33].

# F  Extended Details of Experiments

In this section, we conduct follow-up experiments to enhance the effectiveness of AE-NMCF. The statistics of the data sets are summarized in Table III, and the implementation details are described briefly below:

- We deploy the competing models using the best publicly available implementation with Python 3.8 on an Ubuntu server with a Core i9-1090K 3.7 GHz and 128 GB memory.
- For AE-NMCF, we set the number of iterations and the stopping threshold $\epsilon$ as 500 and 5 to guarantee convergence. The hyperparameters T and $\gamma$ are set in Section F.6.
- For each dataset, we reshape the response logs to the scoring matrix and utilize a 80%/20% train/test split. All models' performances are averaged over 5 repeated trials to ensure fairness.

## F.1  Statistical Hypothesis Test

We first conduct the hypothesis test for the student performance prediction and the knowledge proficiency estimation. Table IV shows the details of the paired $t$-test results, where each entry denotes the $p$-value of the AE-NMCF with the baseline in terms of a given metric. According to Table IV, AE-NMCF shows a significant difference at the 5% level with the baselines in most cases. We can conclude that the prediction (estimation) performance of AE-NMCF is significantly different from that of the competitive models.

Table III: The statistics of data sets

| Statistics | Data Set | | | | | |
| --- | --- | --- | --- | --- | --- | --- |
| | FrcSub | Junyi-s | Quanlang-s | SLP-Bio-s | SLP-His-s | SLP-Eng |
| # Student | 536 | 1,091 | 50 | 100 | 1057 | 360 |
| # Exercise | 20 | 9 | 107 | 129 | 326 | 362 |
| # Knowledge concept | 8 | 9 | 14 | 7 | 14 | 19 |
| Subject | Math | Math | Math | Biology | History | English |
| Relations[1] | many-to-many | one-to-one | one-to-many | one-to-many | one-to-many | one-to-many |
| Sparsity[2] | 0% | 75.03% | 68.67% | 54.92% | 84.28% | 96.92% |

[1] The relationships between knowledge concepts and exercises.
[2] The sparsity of student scoring matrix.

Table IV: Paired $t$-test for the prediction (estimation) results of AE-NMCF with other methods

| Metric | Comparison | | | Data set | | | | | |
| --- | --- | --- | --- | --- | --- | --- | --- | --- | --- |
| | | | | FrcSub | Junyi-s | Quanlang-s | SLP-Bio-s | SLP-His-s | SLP-Eng |
| ACC | AE-NMCF | vs. | NMF | 0.000* | 0.006* | 0.000* | 0.000* | 0.000* | 0.001* |
| | | | MCF-Gra | 0.000* | 0.000* | 0.000* | 0.000* | 0.000* | 0.000* |
| | | | MCF-New | 0.000* | 0.000* | 0.000* | 0.000* | 0.000* | 0.000* |
| | | | GNMF | 0.000* | 0.008* | 0.000* | 0.000* | 0.000* | 0.000* |
| | | | NMMF | 0.000* | 0.040* | 0.000* | 0.000* | 0.000* | 0.001* |
| | | | SNMCF | 0.002* | 0.274 | 0.009* | 0.001* | 0.238 | 0.041* |
| | | | DINA | 0.023* | 0.000* | 0.000* | 0.000* | 0.000* | 0.000* |
| | | | DIRT | 0.000* | 0.000* | 0.000* | 0.000* | 0.000* | 0.000* |
| | | | DeepCDF | 0.001* | 0.000* | 0.002* | 0.002* | 0.000* | 0.000* |
| | | | QRCDM | 0.140 | 0.009* | 0.000* | 0.000* | 0.049* | 0.047* |
| RMSE | AE-NMCF | vs. | NMF | 0.000* | 0.003* | 0.000* | 0.000* | 0.000* | 0.000* |
| | | | MCF-Gra | 0.000* | 0.000* | 0.000* | 0.000* | 0.000* | 0.000* |
| | | | MCF-New | 0.000* | 0.001* | 0.000* | 0.000* | 0.000* | 0.000* |
| | | | GNMF | 0.000* | 0.003* | 0.000* | 0.000* | 0.000* | 0.000* |
| | | | NMMF | 0.000* | 0.334 | 0.000* | 0.000* | 0.000* | 0.012* |
| | | | SNMCF | 0.072 | 0.866 | 0.022* | 0.001* | 0.060 | 0.014* |
| | | | DINA | 0.000* | 0.000* | 0.000* | 0.000* | 0.000* | 0.000* |
| | | | DIRT | 0.000* | 0.019* | 0.000* | 0.000* | 0.000* | 0.000* |
| | | | DeepCDF | 0.248 | 0.002* | 0.000* | 0.000* | 0.000* | 0.000* |
| | | | QRCDM | 0.060 | 0.020* | 0.000* | 0.000* | 0.193 | 0.435 |
| KRC ($r_c$) | AE-NMCF | vs. | SNMCF | 0.000* | 0.000* | 0.005* | 0.000* | 0.000* | 0.000* |
| | | | DINA | 0.000* | 0.000* | 0.000* | 0.000* | 0.000* | 0.441 |
| | | | DIRT | 0.000* | 0.008* | 0.001* | 0.000* | 0.000* | 0.001* |
| | | | DeepCDF | 0.000* | 0.001* | 0.010* | 0.001* | 0.000* | 0.006* |
| | | | QRCDM | 0.009* | 0.012* | 0.226 | 0.023* | 0.009* | 0.141 |

* Significant difference at the 5% level.

## F.2 Nemenyi Test

We conduct the Nemenyi test [39] to present the comparison of the proposed AE-NMCF model with the baseline approaches. The Nemenyi test shows the differences between the average ranks among all the compared methods, and any two of which are significantly different if their average ranks differ by at least one crucial difference (5% in this paper). As illustrated in Figure II, it is obvious that the AE-NMCF model performs the best in terms of ACC and RMSE, which demonstrates its effectiveness in student performance prediction.

## F.3 A Case Study for Diagnostic Comparison

To get a sense of diagnostic improvement of AE-NMCF for data mining techniques (compared with SNMCF), we present the diagnostic results for case students on FrcSub, Quanlang-s, and Junyi-s, which covers all typical knowledge-exercise relationships. [4]

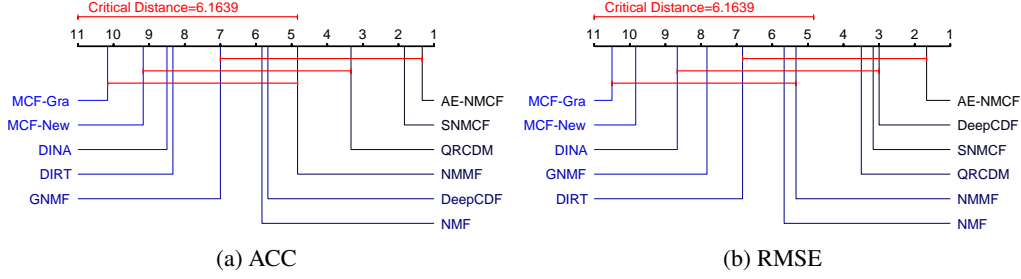

(a) ACC                                                 (b) RMSE

Figure II: The CD diagrams of all the methods in terms of ACC and RMSE.

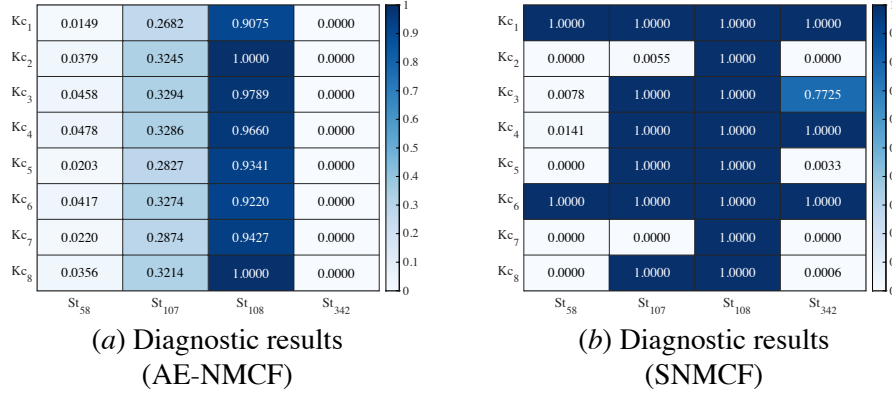

(*a*) Diagnostic results          (*b*) Diagnostic results
(AE-NMCF)                         (SNMCF)

Figure III: Case students' cognitive diagnostic results (AE-NMCF vs. SNMCF) on FrcSub.

**FrcSub.** Figure III compares four case students' diagnostic results from AE-NMCF and SNMCF, respectively, where each numerical value is a student's knowledge proficiency on a specific knowledge concept. In addition, we show the corresponding scoring matrix in Table V. For student $St_{108}$, we can observe that both AE-NMCF and SNMCF give suitable diagnostic results since she answers all the exercises correctly. However, for students $St_{58}$ and $St_{342}$, Table V shows that they only give the right answer to $Ex_9$ and fail in the remaining exercises, which means that $St_{58}$ and $St_{342}$ can not grasp all knowledge concepts. The diagnostic result given by the AE-NMCF model confirms this fact, while SNMCF gives a confusing result. In addition, the response log of $St_{107}$ indicates that she needs to continuously make progress on most of the knowledge concepts to improve proficiency levels, but SNMCF argues that $St_{107}$ has mastered most of the knowledge concepts, which does not square with the facts. In summary, the diagnostic outputs provided by AE-NMCF align with our expectations.

Table V: The corresponding scoring matrix on FrcSub

| Exercise | Student | | | | Exercise | Student | | | |
|---|---|---|---|---|---|---|---|---|---|
| | $St_{58}$ | $St_{107}$ | $St_{108}$ | $St_{342}$ | | $St_{58}$ | $St_{107}$ | $St_{108}$ | $St_{342}$ |
| $Ex_1$ | 0 | 1 | 1 | 0 | $Ex_{11}$ | 0 | 0 | 1 | 0 |
| $Ex_2$ | 0 | 1 | 1 | 0 | $Ex_{12}$ | 0 | 0 | 1 | 0 |
| $Ex_3$ | 0 | 1 | 1 | 0 | $Ex_{13}$ | 0 | 0 | 1 | 0 |
| $Ex_4$ | 0 | 1 | 1 | 0 | $Ex_{14}$ | 0 | 0 | 1 | 0 |
| $Ex_5$ | 0 | 0 | 1 | 0 | $Ex_{15}$ | 0 | 0 | 1 | 0 |
| $Ex_6$ | 0 | 1 | 1 | 0 | $Ex_{16}$ | 0 | 0 | 1 | 0 |
| $Ex_7$ | 0 | 0 | 1 | 0 | $Ex_{17}$ | 0 | 0 | 1 | 0 |
| $Ex_8$ | 0 | 1 | 1 | 0 | $Ex_{18}$ | 0 | 0 | 1 | 0 |
| $Ex_9$ | 1 | 0 | 1 | 1 | $Ex_{19}$ | 0 | 0 | 1 | 0 |
| $Ex_{10}$ | 0 | 0 | 1 | 0 | $Ex_{20}$ | 0 | 0 | 1 | 0 |

**Quanlang-s.** Figure IV shows three case students' knowledge proficiency based on radar charts. To facilitate comparison, we also label the student's answer accuracy rates (the ratio of correctly

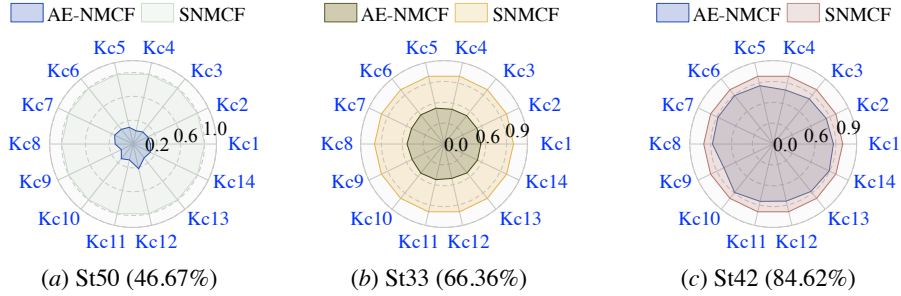

|                          |                          |                          |
|--------------------------|--------------------------|--------------------------|
| (*a*) St50 (46.67%)      | (*b*) St33 (66.36%)      | (*c*) St42 (84.62%)      |

Figure IV: Diagnosis results of three case students between AE-NMCF and SNMCF on Quanlang-s.

answering all exercises), e.g., $46.67\%$ for student $St_{50}$. Intuitively, the proficiency levels of $St_{42}$ should be the highest because of the top accuracy rate, and $St_{50}$ is at the lowest level accordingly. However, SNMCF gives an extreme estimation, which overestimates the ability of $St_{50}$ (or $St_{33}$), and consequently, the cognitive diagnostic ability is limited. Instead, the proposed model gives reasonable results. We conclude that AE-NMCF provides richer information on the diagnosis than SNMCF.

**Junyi-s.** Different from FrcSub, there is substantial missing data in the scoring matrix for Junyi-s, with only $24.97\%$ of its entries observed. Given that the relationship between the knowledge concepts and exercises is one-to-one, we show each knowledge proficiency (provided by AE-NMCF and SNMCF, respectively) with its corresponding answer record of three case students in Figure V, where each subgraph consists of two parts – the response (left) and the knowledge proficiency (right). From Figure V, we have the following observations:

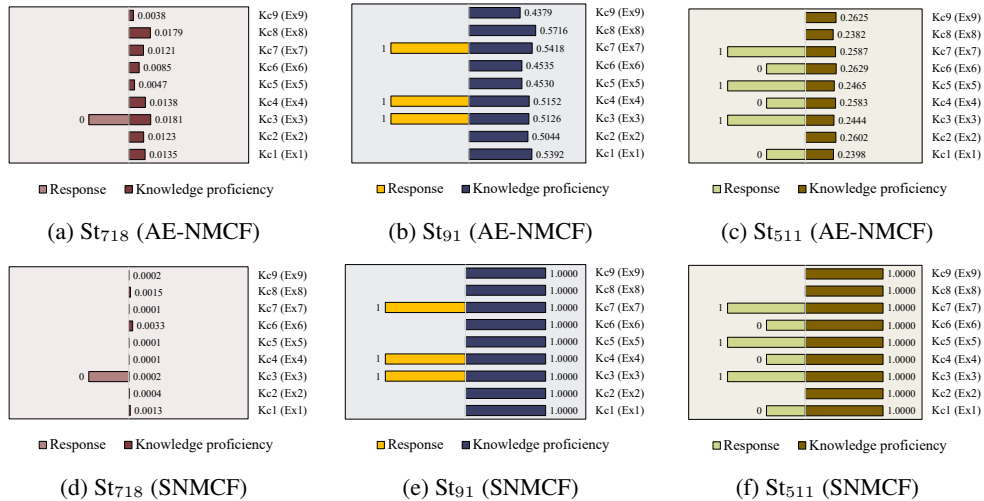

Figure V: Case students' cognitive diagnostic results and the corresponding answer record on Junyi-s.

- First, Figure Va and Figure Vd illustrate that both AE-NMCF and SNMCF provide reasonable diagnostic results since student $St_{718}$ has only a wrong answer record, which lacks far too much information available for diagnosis.
- Second, it can be seen from Figure Vb and Figure Ve that although student $St_{91}$ responds correctly to the given exercises (i.e., $Ex_3$, $Ex_4$, and $Ex_7$), there are still some exercises that $St_{91}$ has never answered before (e.g., $Ex_1$). However, the SNMCF model asserts that the student has completely mastered all knowledge concepts. In contrast, AE-NMCF makes more sense than SNMCF because the new model considers the uncertainty of the missing values of the unanswered exercises.
- Finally, for student $St_{511}$ (see Figure Vc and Figure Vf), we observe that the SNMCF model still gives more illogical diagnostic results than AE-NMCF because the response log shows that $St_{511}$ makes mistakes in some exercises (e.g., $Ex_4$), which indicates that she needs to timely learn the corresponding knowledge concepts (e.g., $Kc_4$).

### F.4 Cognitive Diagnosis Visualization

We proceed to visualize and investigate the diagnostic results of a student as a case study, which provides useful insight into the estimation outcomes of the proposed model. Figure VI displays the student's knowledge proficiency with the corresponding answers on Quanlang-s. As observed, AE-NMCF gives interpretative and meaningful diagnostic results, based on which the student can determine her strengths and shortcomings. For example, the student has a good grasp of all knowledge concepts except for $Kc_{12}$ (*exponentiation of rational numbers*). Observing her responses related to $Kc_{12}$, we notice that the student only tries very few relevant exercises. It suggests a timely study of $Kc_{12}$ for the student. Based on this visualization, AI-based tutoring systems could provide her with personalized remedy plans for improvement.

However, we see that the diagnostic result of $Kc_9$ is overoptimistic, not only because she made many mistakes in the related exercises but also due to her low proficiency in the prerequisite knowledge concepts (e.g., $Kc_7$). Recognizing this limitation, an intuitive work-around is to exploit the knowledge prerequisite structure for AE-NMCF to attenuate this problem.

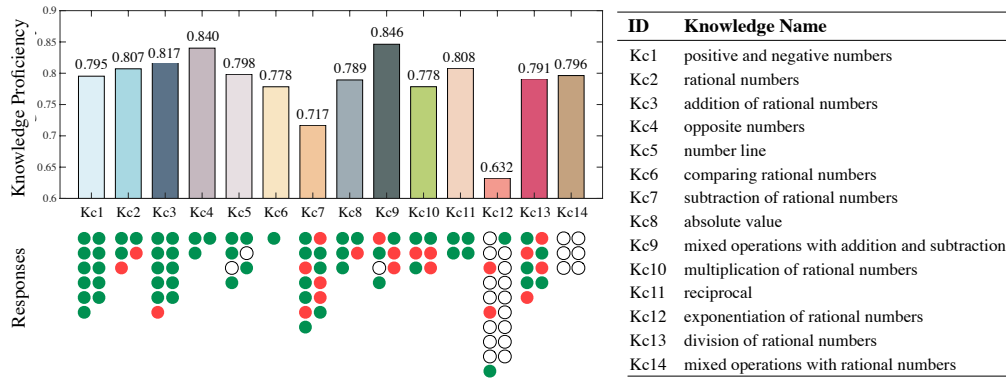

Figure VI: Diagnosis visualization of a case student on Quanlang-s via AE-NMCF. The bottom left shows her responses to related exercises. The circles with green (red) colors represent right (wrong) responses, and the hollow circles denote the absent responses.

### F.5 Comparison of the Step-Size Search Methods

As noted earlier, the "*Armijo* rule along the projection arc" (*Armijo* rule) is another step-size solution. In this section, we show the compared performance between the *Lipschitz* search and the *Armijo* search on FrcSub, Junyi-s, and Quanlang-s, which covers all types of knowledge-exercise relationships. We first check the convergence in Figure VII, which sees that both the search solutions converge to a stationary point; however, the *Armijo* search at first quickly decreases the objective function value but slows down in sequence, which takes more time to converge.

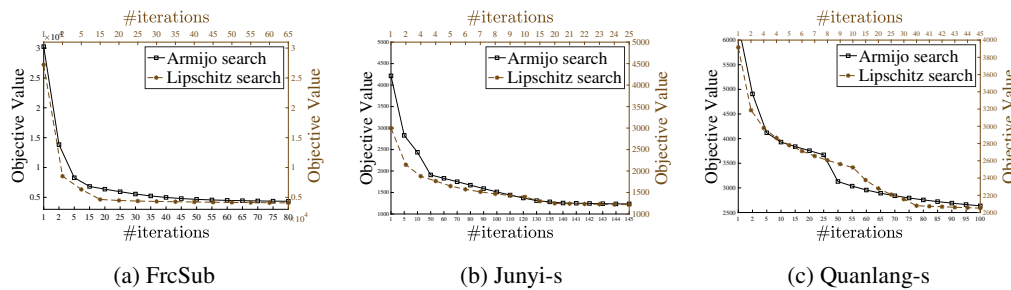

| (a) FrcSub | (b) Junyi-s | (c) Quanlang-s |

Figure VII: The number of iterations vs. objective values for the *Lipschitz* search and *Armijo* search on FrcSub, Junyi-s, and Quanlang-s.

Furthermore, we fix the number of iterations (the smallest one of the two strategies) and present the compared performance in Table VI. We observe that all the methods exhibit similar performance for

Table VI: The comparative results between the *Lipschitz* search and *Armijo* search

| Dataset | Strategy | #iterations | Time (minutes) | Objective value | ACC | $r_c$ |
|---|---|---|---|---|---|---|
| FrcSub | *Armijo* search | 60 | 525.41 | 4496.3772 | 0.8307 | 0.8889 |
| | *Lipschitz* search | | 3.83 | 4100.9518 | 0.8344 | 0.8951 |
| Junyi-s | *Armijo* search | 25 | 425.02 | 2134.1798 | 0.7477 | 0.6205 |
| | *Lipschitz* search | | 0.46 | 1215.7141 | 0.7383 | 0.7243 |
| Quanlang-s | *Armijo* search | 45 | 172.70 | 3238.7009 | 0.7127 | 0.5942 |
| | *Lipschitz* search | | 0.73 | 2050.8577 | 0.7336 | 0.6438 |

student cognitive modeling, while given the same number of iterations, the *Lipschitz* search achieves the fastest convergence while maintaining a relatively small objective function value.

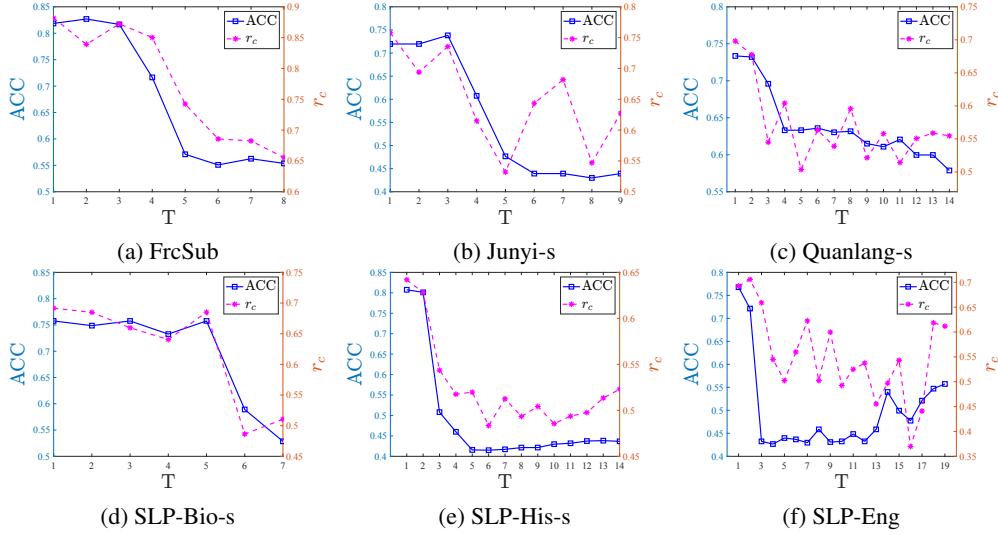

Figure VIII: Sensitivity analysis of parameter T on the data sets.

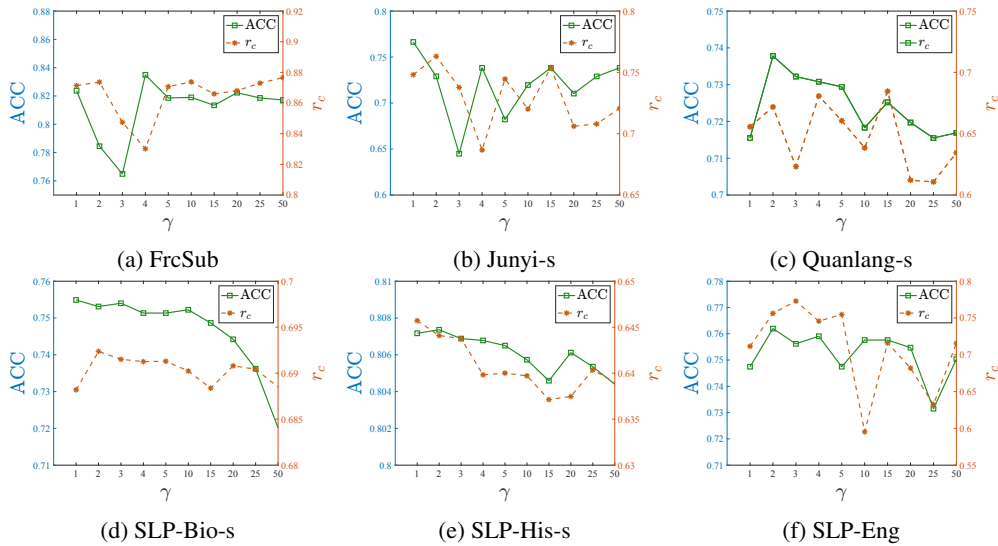

Figure IX: Sensitivity analysis of parameter $\gamma$ on the data sets.

### F.6 Parameter Sensitivity Analysis

Finally, there are two parameters in the AE-NMCF model: (1) the number of latent factors T (i.e., the rank of the nonnegative matrix co-factorization) and (2) the regularization parameter $\gamma$. Since T leads a role in achieving the approximation effect, we begin by discussing T, followed by $\gamma$.

**Effect of parameter** T. We use the grid search rule to tune the value of parameter $T = \min\{N, M, K\}$, and consider the effect in terms of ACC and $r_c$. The results are summarized in Figure VIII. It can be seen from the figure that as the value of T increases, ACC and $r_c$ share a similar decreasing tendency. Therefore, we choose the value that balances the two types of tasks, i.e., we set $T = 3, 3, 2, 1, 1, 1$, for FrcSub, Junyi-s, Quanlang-s, SLP-Bio-s, SLP-His-s, and SLP-Eng respectively as the tuning results.

**Effect of parameter** $\gamma$. Based on the best T value, we proceed to find the best value for $\gamma$, which controls the degree of avoiding the ill-posed problem for **B**. For all the data sets, we perform the grid search with the range of $\{1, 2, 3, 4, 5, 10, 15, 20, 25, 50\}$. By observing the results in Figure IXa, we can see that the ACC ($rc$) leads a drop at the beginning, followed by a sharp rise after $\gamma = 3$ ($\gamma = 4$), and then slightly fluctuates in the sequence. Therefore, we set $\gamma = 10$ for FrcSub. For Junyi-s and Quanlang-s, considering the fluctuation for the $r_c$ value, we choose $\gamma = 2, 4$ respectively to avoid too much regularization. Similarly, we use $\gamma = 2, 1, 3$ as the tuning result for SLP-Bio-s, SLP-His-s, and SLP-Eng, respectively.

